# Learning the Latent Causal Structure for Modeling Label Noise

**Yexiong Lin    Yu Yao    Tongliang Liu**[†]
Sydney AI Centre, The University of Sydney

## Abstract

In label-noise learning, the noise transition matrix reveals how an instance transitions from its clean label to its noisy label. Accurately estimating an instance's noise transition matrix is crucial for estimating its clean label. However, when only a noisy dataset is available, noise transition matrices can be estimated only for some "special" instances. To leverage these estimated transition matrices to help estimate the transition matrices of other instances, it is essential to explore relations between the matrices of these "special" instances and those of others. Existing studies typically build the relation by explicitly defining the similarity between the estimated noise transition matrices of "special" instances and those of other instances. However, these similarity-based assumptions are hard to validate and may not align with real-world data. If these assumptions fail, both noise transition matrices and clean labels cannot be accurately estimated. In this paper, we found that by learning the latent causal structure governing the generating process of noisy data, we can estimate noise transition matrices without the need for similarity-based assumptions. Unlike previous generative label-noise learning methods, we consider causal relations between latent causal variables and model them with a learnable graphical model. Utilizing only noisy data, our method can effectively learn the latent causal structure. Experimental results on various noisy datasets demonstrate that our method achieves state-of-the-art performance in estimating noise transition matrices, which leads to improved classification accuracy. The code is available at: `https://github.com/tmllab/2024_NeurIPS_CSGN`.

## 1  Introduction

Supervised learning relies on annotated large-scale datasets, which can be both time-consuming and costly to create. Although several existing annotation methods offer cost-effective alternatives, such as online queries [5], crowdsourcing [52], and image engines [31], the datasets obtained by these methods are imperfect. The labels of these datasets usually contain errors. These noisy labels would be harmful to deep neural networks because the network can memorize noisy labels easily [59, 13, 3] and lead to the degradation of classification accuracy.

Modeling label noise using noise transition matrices plays an important role in label-noise learning [36, 40, 30, 15]. Let $Y$, $\boldsymbol{X}$, and $\tilde{Y}$ denote the variables for the clean label, the instance, and the noisy label, respectively. The noise transition matrix for an instance $\boldsymbol{x}$ can be represented by $p(\tilde{Y}|Y, \boldsymbol{X} = \boldsymbol{x})$, which reveals the probability that, given an instance, its latent clean label is transited to the observed noisy label. Given these transition matrices, a classifier trained on noisy data by leveraging these matrices can be consistent with the optimal classifier trained on clean data [36, 40, 50].

---

[†]Correspondence to Tongliang Liu (tongliang.liu@sydney.edu.au).

However, noise transition matrices are generally unknown and need to be estimated. When only a noisy dataset is available, only the noise transition matrices of some "special" instances can be estimated. For example, if instances belong to a clean class with a probability of one (known as anchor points), their noise transition matrices can be estimated [50]. To utilize these learned transitions to help in estimating others, it is crucial to establish relations between the transition matrices of these "special" instances and those of the other instances. Existing studies usually build this relation by proposing similarity assumptions for noise transition matrices across different instances. For example: 1). the noise transition matrices of instances within the same class are identical [36, 40, 50]; 2). the noise transition matrices of instances lying in the same manifold are identical [8]; 3). the noise transition matrices are the same for the closest three instances within the same clean class [61, 37]. However, these predefined assumptions may not hold in real-world cases. Moreover, verifying these assumptions with only noisy data is challenging. If these assumptions are violated, it can lead to estimation errors in both transition matrices and clean labels.

In this paper, to establish relations between the transition matrices of these "special" instances and those of the others without relying on predefined similarities, we propose exploring the latent causal structure that generates noisy data. By understanding this structure, the relations among noise transition matrices across instances would be implicitly captured. Subsequently, the noise transition matrices of other instances can be estimated by making use of the estimated noise transition matrices of the "special" instances. To offer some intuition, in Fig. 1, we illustrate two noisy examples in the benchmark noisy dataset CIFAR-10N. Suppose there is a latent variable for the "presence of fur", and an annotator is biased by the "presence of fur". If the first "cat" image containing "fur" is mislabeled as "dog" by this annotator, then a second "cat" image also containing "fur" will likely be

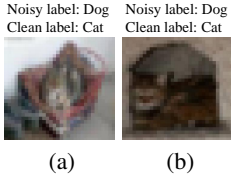

Noisy label: Dog  Noisy label: Dog
Clean label: Cat  Clean label: Cat

(a)             (b)

Figure 1: The pictures contain the same noisy labels.

mislabeled as "dog" with high probability by the same annotator. This leads us to understand that the noise transition matrices for the two images are similar when their causal variables have the same value. In essence, *the relation between noise transition matrices across these two instances is established through the causal structure.* Such insights have driven us to develop an algorithm designed to learn the latent causal structure, thereby enabling the accurate estimation of noise transition matrices for new instances based on existing ones.

There are a few studies [55, 10] that tackle the label-noise learning problem by modeling the data-generating process with a generative model and leveraging causality. However, they may fail to capture the relation among noise transition matrices as the data-generating process proposed by these methods is restrained. Specifically, as shown in Fig. 2a, current methods assume that the instance $X$ is a direct cause of the noisy label $\tilde{Y}$ in the data-generating process. However, this assumption does not hold for many machine learning datasets consisting of perceptual data such as images, videos, and natural language. In these datasets, the latent causal variables such as shape, color, and semantic concepts are the direct causes of the noisy label $\tilde{Y}$ rather than the perceptual data $X$ itself. As previously mentioned, the mislabeling of an image of a "cat" as a "dog" could be due to the latent causal variable "presence of fur". Two images generated under similar latent variable values should have similar transition matrices. Therefore, it is important to consider and learn these latent causal variables in the data-generating process. Motivated by this insight, we propose to incorporate latent causal variables in the data-generating process.

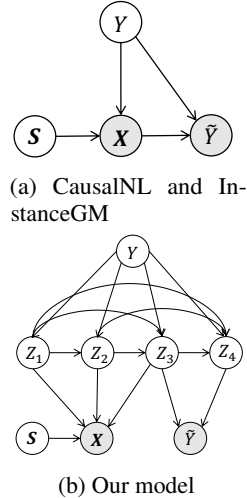

(a) CausalNL and InstanceGM

(b) Our model

Figure 2: Noisy data-generating processes assumed by different methods. $S$ is the variable unrelated to the clean label $Y$.

As shown in Fig. 2b, the observed instance $X$ is generated by two sets of latent variables: $S$ which are not influenced by the clean label $Y$ and $Z$ which are caused by the clean label $Y$. Furthermore, we allow for causal dependencies among the variables in $Z$. We assume that the cause of the noisy label $\tilde{Y}$ also depends on the clean label $Y$. For example, the latent causal variable "presence of fur" influences the noisy label $\tilde{Y}$ and depends on the clean label (e.g., cat or dog), which satisfies this assumption. To learn the data-generating process, we employ a learnable graphical model that models all latent causal variables and their causal relations. Once the model is trained, it can infer the values of latent causal variables for

different images, thereby implicitly capturing the relations among noise transition matrices across instances. Theoretically, we explain the sufficient conditions for identifying the latent causal variables in Appendix B. Empirically, our method achieves state-of-the-art performance in estimating the noise transition matrices and predicting clean labels.

## 2 Related Work

**Modeling the Noise Transition Matrices**    Modeling label noise by noise transition matrices plays an important role for label-noise learning. Some methods [36, 40, 50, 56, 30] assume the noise transition matrix is *instance-independent*, i.e., $p(\tilde{Y} = i|Y = j, \boldsymbol{X} = \boldsymbol{x}) = p(\tilde{Y} = i|Y = j)$. The instance-independent transition matrix is identifiable under some mild conditions. For example, the instance-independent transition matrix can be estimated using anchor points, where the clean class posterior probability of a class is one [36, 40, 50, 56]. Li *et al.* [30] assume that the clean class-posterior distribution is *sufficiently scattered* and learn the instance-independent transition matrix by minimizing the volume of the transition matrix.

To learn instance-dependent transition matrices, existing methods often rely on predefined statistical dependencies to explain how noisy labels are generated. Xia *et al.* [49] propose to linearly decompose an instance into features (or "parts"), assuming that the noisy label depends on all these features. Based on this assumption, they estimate instance-dependent transition matrices by combining the transition matrices associated with these features. Cheng *et al.* [8] assume that noisy labels are dependent on the data manifold, which leads to a manifold-regularized method for learning transition matrices. Unlike these methods, our proposed method automatically discovers dependencies by identifying latent variables that causally influence noisy labels. Furthermore, our method can be viewed as a generalization of Xia *et al.*'s work [49]. Specifically, rather than assuming that noisy labels depend on linearly decomposed features, our method employs a nonlinear model and leverages causal representation learning [38] to infer the features responsible for generating the noisy labels. CausalNL [55] and InstanceGM [10] also leverage causality to learn transition matrices. However, the data-generating processes assumed by these methods can be constrained, as they do not account for the latent causal relationships and variables considered by our method. Other methods for learning with noisy labels are discussed in Appendix G.

**Causal Representation Learning**    Causal representation learning [43] aims to identify latent causal variables from observations. The generating process for observations from the latent variables is typically non-linear. Previous research has established that identifying latent variables in an unsupervised manner is generally infeasible [20, 39]. Recently, the focus has shifted to weakly-supervised or self-supervised methods that incorporate additional supervised information. For example, methods utilizing temporal information [44, 18, 19], auxiliary variables that cause the latent variables [21, 23], and multiview information [6] have been employed to learn causal representations.

## 3 Learning the Latent Causal Structure for Generation of Noisy Data

**A Data-Generating Process**    Let $\boldsymbol{X}$ and $\tilde{Y}$ denote the observed variables of instances and noisy labels, respectively. The observed variable $\boldsymbol{X}$ is generated by a subset of causal variables, $\boldsymbol{Z}$, along with other variables $\boldsymbol{S}$. Similarly, the noisy label $\tilde{Y}$ is also generated by a subset of the causal variables $\boldsymbol{Z}$. The causal variables in $\boldsymbol{Z}$ can exhibit causal dependence, meaning some variables are effects of others. The influence of other variables on a causal variable $Z_i$ can be represented as $Z_i := f_{\boldsymbol{Z}}(pa(Z_i), N_i)$, where $f_{\boldsymbol{Z}}$ is the causal mechanism, $pa(Z_i)$ denotes all the causes of $Z_i$, and $N_i$ is the corresponding latent noise variable. The latent causal structure, which includes the latent causal variables $\boldsymbol{Z}$ and the causal dependencies among them, can be represented by a Directed Acyclic Graph (DAG). Given that this latent causal structure is unknown, our aim is to learn the causal mechanisms that generate both the observed instances and the noisy labels.

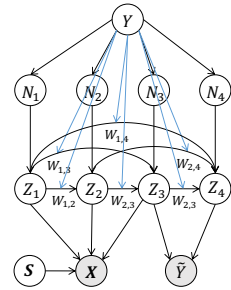

Figure 3: An illustration of the noisy data-generating process with 4 latent variables.

The data-generating process is illustrated in Fig. 3. The latent causal variable $Z_i$ is generated by its parent variables, denoted as $pa(Z_i)$, along

with a corresponding latent noise variable $N_i$. The instance $\boldsymbol{X}$ and the noisy label $\tilde{Y}$ are generated by different subsets of latent variables. The black arrow in the graph indicates the causal direction in the structural causal model, while the blue arrow indicates the weights of the edges vary with the clean label $Y$. The instance $\boldsymbol{X}$ and the noisy label $\tilde{Y}$ are dependent because they are generated by the common cause variables $\boldsymbol{Z}$ [41, 42], which implies that the transition matrices in our model are instance-dependent.

Let $pa(\boldsymbol{X})$ and $pa(\tilde{Y})$ denote the sets of causal variables for generating $\boldsymbol{X}$ and $\tilde{Y}$, respectively. Let $\boldsymbol{f_X}$ and $\boldsymbol{f_{\tilde{Y}}}$ denote nonlinear mixing functions. Let $\boldsymbol{\varepsilon_X}$ and $\boldsymbol{\varepsilon_{\tilde{Y}}}$ denote independent noise variables with probability density functions $p_{\boldsymbol{\varepsilon_X}}(\boldsymbol{\varepsilon_X})$ and $p_{\boldsymbol{\varepsilon_{\tilde{Y}}}}(\boldsymbol{\varepsilon_{\tilde{Y}}})$. The generating process of the noisy data can be modeled as a structural causal model (SCM) as follows:

$$
\begin{aligned}
N_i &:\sim \mathcal{N}(\mu_{N_i}(Y), \sigma_{N_i}^2(Y)), & Z_i &:= \boldsymbol{W}_i^T(Y)\boldsymbol{Z} + N_i, \\
\boldsymbol{X} &:= \boldsymbol{f_X}(pa(\boldsymbol{X}), \boldsymbol{S}) + \boldsymbol{\varepsilon_X}, & \tilde{Y} &:= \boldsymbol{f_{\tilde{Y}}}(pa(\tilde{Y})) + \boldsymbol{\varepsilon_{\tilde{Y}}},
\end{aligned}
\tag{1}
$$

where $\mathcal{N}(\mu_{N_i}(Y), \sigma_{N_i}^2(Y))$ represents the Gaussian distribution with mean $\mu_{N_i}(Y)$ and variance $\sigma_{N_i}^2(Y)$. The matrix $\boldsymbol{W}(Y)$ represents the causal association among causal variables. The conditional distribution for the causal variables $\boldsymbol{Z}$ given the clean label $Y$ can be denoted as $p_{\boldsymbol{W}, \boldsymbol{\mu_N}, \boldsymbol{\sigma_N^2}}(\boldsymbol{Z}|Y)$.

Unlike previous methods [55, 10], we do not assume the latent causal variables are independent. We allow causal relations between different causal variables in $\boldsymbol{Z}$, which is more general. For example, in an image, the causal variable associated with the "presence of the sun" can influence the causal variable related to "brightness". Moreover, we allow both the latent noise variable $\boldsymbol{N}$ and the weights of causal association $W_{i,j}(Y)$ to be different across different clean labels. This variability enables latent causal variables to exert varying degrees of influence on noisy labels in different classes. For example, mislabeling a "cat" as a "dog" might be influenced by the causal variable "presence of fur", whereas mislabeling a "house" as a "car" is likely less influenced by "presence of fur" and more by other causal variable, such as the causal variable related to "shape". The data-generating process in our model is nonlinear. Specifically, the instance $\boldsymbol{X}$ and the noisy label $\tilde{Y}$ are generated through nonlinear functions. However, to establish sufficient conditions for identifiability, the causal relations within $\boldsymbol{Z}$ are assumed to be linear. This assumption is further elaborated in Appendix B.

**Intuition about Learning Latent Data-Generating Process**  Existing theories in causal representation learning [53, 38] suggest that the data-generating process can be efficiently learned with the aid of additional supervised information. In label-noise learning, this additional supervised information typically requires some clean examples. We follow the previous methods [55, 10] that assume a subset of clean examples can be selected from the noisy training data using current techniques [28]. With some supervised information derived from these selected clean examples, learning the data-generating process becomes feasible.

Here, we provide some intuition about the core idea behind the existing identifiability result [38], explaining why the data-generating process can be effectively learned. Central to this understanding is the realization that the parameters governing this data-generating process are not unique to individual examples but are shared across them. Specifically, examples from the same class share the same data-generating process. The essence of learning the data-generating process lies in learning these shared parameters. When these parameters are shared across different examples, the total number of parameters does not increase as more examples are provided. Instead, these parameters are selected to fit the examples as well as possible. For example, in a linear model, as more data are provided, the system accumulates more linear equations but maintains a fixed number of parameters. This setup leads to identifiability when the number of equations exceeds the number of parameters, which enables a precise estimation of the parameters. With a set of selected clean examples along with their noisy labels, one can calculate the most probable parameter value for generating these examples. This approach reduces the uncertainty of the parameter value. In Appendix B, we discuss the specific assumption to make the latent generating process of noisy data identifiable.

## 3.1 Methodology

We propose a model-based method which learns the latent Causal Structure for the Generation of Noisy data (CSGN). The workflow of our method is illustrated in Fig. 4. A classification network $\boldsymbol{g}_Y$ is used to model the distribution $q_\psi(Y|\boldsymbol{X})$. An encoder $\boldsymbol{g}_{\boldsymbol{Z},\boldsymbol{S}}$ is employed to model the distribution

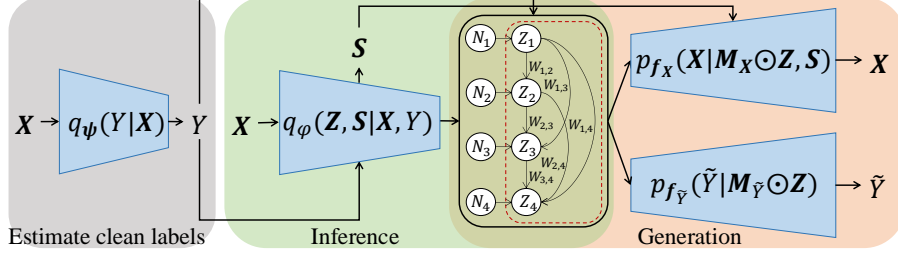

Figure 4: A workflow of our method. In the inference stage, a classification network is used to learn the clean labels of instances; An encoder is used to learn the causal variables. In the generation stage, different subsets of causal variables used to generate instances and noisy labels are selected by masking. Two decoders are used to generate the instances and noisy labels.

$q_{\varphi}(\boldsymbol{Z}, \boldsymbol{S} | \boldsymbol{X}, Y)$. Two vectors $\boldsymbol{M_X}$ and $\boldsymbol{M_{\tilde{Y}}}$ are employed to mask some causal variables $\boldsymbol{Z}$. These vectors select different causal variables for the generation of the instance $\boldsymbol{X}$ and the noisy label $\tilde{Y}$. The distribution $p_{\boldsymbol{f_X}}(\boldsymbol{X} | \boldsymbol{M_X} \odot \boldsymbol{Z}, \boldsymbol{S})$ is modeled by a decoder $\boldsymbol{f_X}$ for generating instances. The distribution $p_{\boldsymbol{f_{\tilde{Y}}}}(\tilde{Y} | \boldsymbol{M_{\tilde{Y}}} \odot \boldsymbol{Z})$ is modeled by a decoder $\boldsymbol{f_{\tilde{Y}}}$ for generating noisy labels. These encoders and decoders can be effectively learned within the Variational Autoencoder (VAE) framework [24].

**Warmup** Existing methods [55, 10] that employ generative models require some clean examples. Similarly, our method also requires clean examples, where their clean labels provide additional supervised information to learn latent causal variables and their causal relations. In settings involving label-noise learning, the clean label is typically unknown. We only have a noisy dataset $\tilde{\mathcal{D}} = \{(\boldsymbol{x}^{(i)}, \tilde{y}^{(i)})\}_{i=1}^{n}$, where $n$ is the number of examples. To obtain the additional supervision information, we follow the existing methods [55, 10] which adopt the small-loss trick [13] to select some clean examples. This method is based on the observation that a classification network trained on noisy data tends to first memorize examples with correct labels before those with incorrect ones. Consequently, during the early stages of training, the losses for examples with correct labels are usually smaller than those for examples with incorrect labels. By analyzing these loss values, we can distinguish clean examples from noisy ones. We adopt a semi-supervised learning approach, as detailed in [4], to train the classification network for selecting clean examples. Further specifics of this approach are provided in Appendix C. After this initial warmup phase, once some clean examples have been identified, we compile a new dataset consisting of instances, noisy labels, and clean labels as $\mathcal{D} = \{(\boldsymbol{x}^{(i)}, \tilde{y}^{(i)}, y^{(i)})\}_{i=1}^{m}$, where $m$ is the size of the dataset.

**Modeling Latent Causal Variables and their Causal Relations** We first model the generating process of the causal variables $\boldsymbol{Z}$ using the SCM in Eq. (1). Given that the relationships between the latent variables are linear functions of their causes, plus some independent Gaussian noise, this linearity implies that the distribution of the latent causal variables $\boldsymbol{Z}$, conditioned on the true label $Y$, also follows a Gaussian distribution. Specifically, the conditional distribution is given by:

$$p_{\boldsymbol{W}, \boldsymbol{\mu_N}, \boldsymbol{\sigma_N^2}}(\boldsymbol{Z} | Y) = \mathcal{N}(\boldsymbol{\mu_Z}, \boldsymbol{\Sigma}), \tag{2}$$

where $\boldsymbol{\mu_Z}$ is the mean vector, and $\boldsymbol{\Sigma}$ is the covariance matrix of the latent variables $\boldsymbol{Z}$. The mean $\boldsymbol{\mu_Z}$ and the covariance $\boldsymbol{\Sigma}$ are determined by the weight matrix $\boldsymbol{W}$ and the parameters of the latent noise variables, specifically $\boldsymbol{\mu_N}$ and $\boldsymbol{\sigma_N^2}$. Note that the weight matrix $\boldsymbol{W}$ encodes the causal relationships among the latent variables $\boldsymbol{Z}$, with each entry representing the strength of the causal effect from one variable to another.

Let $\sigma_{Z_i, Z_j}^2$ denote the element in the $i$-th row and $j$-th column of the covariance matrix $\boldsymbol{\Sigma}$. Given that the generating process of causal variables is modeled by a linear SCM, we can compute the parameters $\mu_{Z_i}$, $\sigma_{Z_i, Z_i}^2$, and $\sigma_{Z_i, Z_j}^2$ for a causal variable $Z_i$ by the following recursion relations [25]:

$$\mu_{Z_i} = \sum_{j \in pa_i} W_{j,i}(Y) \mu_{Z_j} + \mu_{N_i}(Y),$$

$$\sigma_{Z_i, Z_i}^2 = \sum_{j \in pa_i} W_{j,i}^2(Y) \sigma_{Z_j, Z_j}^2 + \sigma_{N_i}^2(Y),$$

$$\sigma_{Z_i, Z_j}^2 = \sum_{k \in pa_j} W_{k,j}(Y) \sigma_{Z_i, Z_k}^2, \quad for \ i \neq j,$$

where $pa_i$ represents the set of indices for the parents of the causal variable $Z_i$. We let $\boldsymbol{W}$, $\boldsymbol{\mu_N}$ and $\boldsymbol{\sigma_N^2}$ be influenced by $Y$. To provide identifiable results, $\boldsymbol{W}$, $\boldsymbol{\mu_N}$ and $\boldsymbol{\sigma_N^2}$ have to be determined by $Y$ [38]. Exploring ways to relax this assumption will be a focus of our future work.

**Modeling the Generation of Observed Variables**    In real-world scenarios, instances and noisy labels may be generated by different subsets of latent causal variables. To accommodate this, two learnable masks are employed to selectively activate different subsets of causal variables for generating instances and noisy labels, respectively. We also utilize distinct decoders to model the generation processes of instances and noisy labels.

Let $\boldsymbol{Z_X}$ and $\boldsymbol{Z_{\tilde{Y}}}$ denote the subsets of causal variables used to generate the instance $\boldsymbol{X}$ and the noisy label $\tilde{Y}$. To select the subsets of causal variables $\boldsymbol{Z_X}$ and $\boldsymbol{Z_{\tilde{Y}}}$, we employ two masks. Specifically, let $\boldsymbol{M_X}$ and $\boldsymbol{M_{\tilde{Y}}}$ denote two vectors for sparsity. They contain learnable parameters that dictate which causal variables are active during the generation process. The masking process is as follows:

$$\boldsymbol{Z_X} = \boldsymbol{M_X} \odot \boldsymbol{Z}, \ \ \boldsymbol{Z_{\tilde{Y}}} = \boldsymbol{M_{\tilde{Y}}} \odot \boldsymbol{Z},$$

where $\odot$ is the element-wise multiplication. To let vectors $\boldsymbol{M_X}$ and $\boldsymbol{M_{\tilde{Y}}}$ act as masks, these vectors are designed to be sparse. This sparsity is encouraged through an L1 regularization loss:

$$\mathcal{L}_M = \|\boldsymbol{M_X}\|_1 + \|\boldsymbol{M_{\tilde{Y}}}\|_1. \tag{3}$$

Note that since parameters in the mask is learnable, our masking method can select either different subsets or the same subsets of variables for generating the instance $\boldsymbol{X}$ and the noisy label $\tilde{Y}$ by optimizing the loss defined in Eq. 5.

Instances and noisy labels are generated through different mechanisms. Thus, we employ two different decoders: $\boldsymbol{f_X}$ for the instances and $\boldsymbol{f_{\tilde{Y}}}$ for the noisy labels. The generating process of these variables is defined as follows:

$$\boldsymbol{X} = \boldsymbol{f_X}(\boldsymbol{Z_X}, \boldsymbol{S}) + \boldsymbol{\varepsilon_X}, \ \ \tilde{Y} = \boldsymbol{f_{\tilde{Y}}}(\boldsymbol{Z_{\tilde{Y}}}) + \boldsymbol{\varepsilon_{\tilde{Y}}},$$

where $\boldsymbol{\varepsilon_X}$ and $\boldsymbol{\varepsilon_{\tilde{Y}}}$ represent independent noise variables, each with its respective probability density functions $p_{\boldsymbol{\varepsilon_X}}(\boldsymbol{\varepsilon_X})$ and $p_{\boldsymbol{\varepsilon_{\tilde{Y}}}}(\boldsymbol{\varepsilon_{\tilde{Y}}})$. The above equations establish the framework for our generative models for instances and noisy labels, which are probabilistic models defined as follows:

$$p_{\boldsymbol{f_X}}(\boldsymbol{X}|\boldsymbol{Z_X}, \boldsymbol{S}) = p_{\boldsymbol{\varepsilon_X}}(\boldsymbol{X} - \boldsymbol{f_X}(\boldsymbol{Z_X}, \boldsymbol{S})),$$
$$p_{\boldsymbol{f_{\tilde{Y}}}}(\boldsymbol{X}|\boldsymbol{Z_{\tilde{Y}}}) = p_{\boldsymbol{\varepsilon_{\tilde{Y}}}}(\tilde{Y} - \boldsymbol{f_{\tilde{Y}}}(\boldsymbol{Z_{\tilde{Y}}})).$$

As shown in Fig. 3, we consider the mechanisms generating instances and noisy labels to be independent. Therefore, the joint generating process of $\boldsymbol{X}$ and $\tilde{Y}$ conditioned on the latent variables can be modeled by the product of their individual probabilities, i.e.,

$$p_{\boldsymbol{f}}(\boldsymbol{X}, \tilde{Y}|\boldsymbol{Z}, \boldsymbol{S}) = p_{\boldsymbol{f_X}}(\boldsymbol{X}|\boldsymbol{Z_X}, \boldsymbol{S})p_{\boldsymbol{f_{\tilde{Y}}}}(\tilde{Y}|\boldsymbol{Z_{\tilde{Y}}}),$$

where $\boldsymbol{f} = \{\boldsymbol{f_X}, \boldsymbol{f_{\tilde{Y}}}\}$ encompasses both decoders, and $\boldsymbol{Z} = \{\boldsymbol{Z_X} \cup \boldsymbol{Z_{\tilde{Y}}}\}$.

We further assume that $p(\boldsymbol{S})$, which is the distribution of the latent variable $\boldsymbol{S}$, follows a standard multivariate normal distribution. Integrating all components, the overall generative model is defined as a probabilistic model parameterized by $\theta = (\boldsymbol{f}, \boldsymbol{W}, \boldsymbol{\mu_N}, \boldsymbol{\sigma_N^2})$ defined as:

$$p_\theta(\boldsymbol{X}, \tilde{Y}, \boldsymbol{Z}, \boldsymbol{S}|Y) = p_{\boldsymbol{f}}(\boldsymbol{X}, \tilde{Y}|\boldsymbol{Z}, \boldsymbol{S})p_{\boldsymbol{W}, \boldsymbol{\mu_N}, \boldsymbol{\sigma_N^2}}(\boldsymbol{Z}|Y)p(\boldsymbol{S}). \tag{4}$$

Here, $p_{\boldsymbol{W}, \boldsymbol{\mu_N}, \boldsymbol{\sigma_N^2}}(\boldsymbol{Z}|Y)$ specifies the distribution of the latent variables conditioned on the true label $Y$, parameterized by the weight matrix $\boldsymbol{W}$ and the parameters of the independent noise, $\boldsymbol{\mu_N}, \boldsymbol{\sigma_N^2}$.

**Optimization**    After the warmup phase, we can construct a new dataset $\mathcal{D} = \{(\boldsymbol{x}^{(i)}, \tilde{y}^{(i)}, y^{(i)})\}_{i=1}^m$, where each instance includes the data $\boldsymbol{x}^{(i)}$, the noisy label $\tilde{y}^{(i)}$, and the corresponding true label $y^{(i)}$. We train our model's parameters using this dataset. Let $p(\boldsymbol{X}, \tilde{Y}, Y)$ denote the underlying joint distribution of the variables $\boldsymbol{X}$, $\tilde{Y}$, and $Y$. In line with prior work [55], we approximate this distribution with $q_{\mathcal{D}}(\boldsymbol{X}, \tilde{Y}, Y)$ using the product of the empirical data distribution and the model's predictive distribution $q_{\mathcal{D}}(\boldsymbol{X}, \tilde{Y}, Y) \approx q_{\tilde{\mathcal{D}}}(\boldsymbol{X}, \tilde{Y})q_\psi(Y|\boldsymbol{X})$ where $q_{\tilde{\mathcal{D}}}(\boldsymbol{X}, \tilde{Y})$ is the empirical distribution derived from the noisy dataset $\tilde{\mathcal{D}}$, and $q_\psi(Y|\boldsymbol{X})$ is the estimated clean class posterior provided by the classification network $\boldsymbol{g_Y}$. To optimize the encoder and parameters in $\theta$ for the generative

model, we maximize the Evidence Lower Bound (ELBO) on the marginal likelihood of the observed data $\mathcal{D}$. The ELBO is formulated as:

$$ELBO = \mathbb{E}_{(\boldsymbol{x},\tilde{y},y)\sim q_{\tilde{\mathcal{D}}}q_\psi} \left[ \mathbb{E}_{(\boldsymbol{z},\boldsymbol{s})\sim q_\varphi} [\log p_{\boldsymbol{f}}(\boldsymbol{x},\tilde{y}|\boldsymbol{z},\boldsymbol{s})] \right.$$
$$\left. - KL(q_\varphi(\boldsymbol{z}|\boldsymbol{x},y)||p_{\boldsymbol{W},\boldsymbol{\mu_N},\boldsymbol{\sigma_N^2}}(\boldsymbol{z}|y)) - KL(q_\varphi(\boldsymbol{s}|\boldsymbol{x},y)||p(\boldsymbol{s})) \right], \qquad (5)$$

where $KL$ denotes the Kullback–Leibler divergence, $q_\varphi(\boldsymbol{z}|\boldsymbol{x},y)$ and $q_\varphi(\boldsymbol{s}|\boldsymbol{x},y)$ are approximate posterior distributions which can be derived by with the encoder $\boldsymbol{g_{Z,S}}$. The derivation of ELBO is detailed in Appendix A. The final loss function used to train the networks is

$$\mathcal{L} = \mathcal{L}_{semi} - \lambda_{ELBO} ELBO + \lambda_M \mathcal{L}_M, \qquad (6)$$

where $\lambda_{ELBO}$ and $\lambda_M$ are hyperparameters. The algorithm of CSGN is given in Alg. 1.

## 4 Experiments

In this section, we report the experiment results of our method. We first compare the effectiveness of the proposed data-generating process with existing methods. We then compare the estimation error of noise transition matrices with other methods and the classification performance of the proposed method with that of state-of-the-art methods on synthetic and real-world noisy datasets. The sensitivity tests of the hyper-parameters are in Appendix H. The results of the ablation study are in Appendix I. The number and the accuracy of the selected clean samples are available in Appendix J. We report the performance of a classification network trained on these clean samples in Appendix K. To verify the effectiveness of our method in recovering the causal graph, we conduct experiments on the synthetic dataset in Appendix L. The visualization of the transition matrices is in Appendix M.

### 4.1 Experiment Setup

**Datasets** We empirically verify the performance of our method on three synthesis datasets, i.e., Fashion-MNIST [51], CIFAR-10 [26], CIFAR-100 [26], and two real-world datasets, i.e., CIFAR-N [48] and Webvision [29]. Fashion-MNIST contains 70,000 28x28 grayscale images with 10 classes total, 60,000 images for training, and 10,000 images for testing. Both CIFAR-10 and CIFAR-100 contain 50,000 training images and 10,000 testing images. The image size is 32x32. CIFAR-10 has 10 classes of images, and CIFAR-100 has 100 classes of images. The three datasets contain clean labels. We corrupted the training data manually according to the instance-dependent noisy label generation method proposed in [49]. CIFAR-N contains CIFAR-10N and CIFAR-100N. CIFAR-10N is a real-world label-noise version of CIFAR-10. It contains human-annotated noisy labels with five different types of noise (Worst, Aggregate, Random 1, Random 2, and Random 3). The corresponding noise rates are 40.21%, 9.03%, 17.23%, 18.12%, and 17.64%. CIFAR-100N is a real-world label-noise version of CIFAR-100. It contains the type of noise (Fine). The corresponding noise rates are 40.20%. Webvision dataset [29] is a large-scale real-world dataset. We follow the previous work [7] to train the model on the first 50 classes of the Google image subset and test the model on the WebVision validation set and the ImageNet ILSVRC12 validation set.

**Implementation** We implement our algorithm using PyTorch and conduct experiments on eight RTX-3090 GPUs. We use a PreAct ResNet-18 [14] as the classification network for Fashion-MNIST [51], CIFAR-10 [26], CIFAR-100 [26], and CIFAR-N [48], an inception-resnet v2 [45] as the classification network for WebVision. More details about the structure of decoders and encoders are in Appendix D. To prevent the accumulation of errors from biased selection, we adopt the approach of using two neural networks to select clean examples for each other, following the approach in previous work [13]. The number of variables in $\boldsymbol{Z}$ is set to 4 in all our experiments. For experiments on Fashion-MNIST, CIFAR-10, CIFAR-100 and CIFAR-N, we employed SGD with a momentum of 0.9 and a weight decay of 0.0005 to optimize the classification network $\boldsymbol{g_Y}$. We used Adam with default parameters to optimize the encoder $\boldsymbol{g_{Z,S}}$, the decoder $\boldsymbol{f_X}$, and the decoder $\boldsymbol{f_{\tilde{Y}}}$ and other parameters $\{\boldsymbol{W},\boldsymbol{\mu_N},\boldsymbol{\sigma_N^2}\}$. The initial learning rate for SGD was set at 0.02 and for Adam at 0.001. Our networks were trained for 200 epochs with a batch size of 64. Both learning rates were reduced by a factor of 10 after 100 epochs. For experiments on WebVision, we changed the weight decay of SGD to 0.001. The initial learning rate for SGD was set at 0.04 and for Adam at 0.004. Other parameters of optimizers remain unchanged. Our networks were trained for 80 epochs with a batch

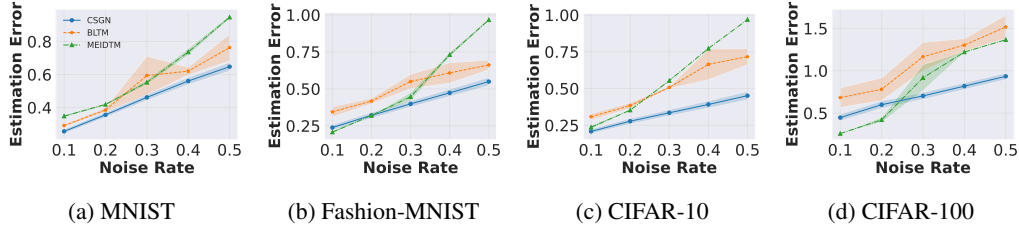

|  | (a) MNIST | (b) Fashion-MNIST | (c) CIFAR-10 | (d) CIFAR-100 |

Figure 5: The estimation error of noise transition matrices. The datasets are MNIST, Fashion-MNIST, CIFAR-10 and CIFAR-100 with the instance-dependent label noise. The error bar for standard deviation in each figure has been shaded.

Table 1: Replace the generative model of CausalNL and InstanceGM with ours. Experiments are on CIFAR-10.

| | CIFAR-10 | | | | |
| --- | --- | --- | --- | --- | --- |
| | IDN-10% | IDN-20% | IDN-30% | IDN-40% | IDN-50% |
| CausalNL | $90.31 \pm 0.09$ | $89.76 \pm 0.08$ | $88.06 \pm 0.53$ | $80.97 \pm 1.99$ | $57.93 \pm 7.53$ |
| CausalNL$'$ | $\mathbf{91.59 \pm 0.26}$ | $\mathbf{91.29 \pm 0.07}$ | $\mathbf{90.44 \pm 0.05}$ | $\mathbf{89.49 \pm 0.31}$ | $\mathbf{88.83 \pm 0.09}$ |
| InstanceGM | $96.07 \pm 0.14$ | $96.00 \pm 0.09$ | $95.95 \pm 0.11$ | $95.99 \pm 0.10$ | $95.81 \pm 0.07$ |
| InstanceGM$'$ | $\mathbf{96.20 \pm 0.07}$ | $\mathbf{96.59 \pm 0.07}$ | $\mathbf{96.45 \pm 0.11}$ | $\mathbf{96.52 \pm 0.11}$ | $\mathbf{96.32 \pm 0.07}$ |

Table 2: Replace the generative model of CausalNL and InstanceGM with ours. Experiments are on CIFAR-100.

| | CIFAR-100 | | | | |
| --- | --- | --- | --- | --- | --- |
| | IDN-10% | IDN-20% | IDN-30% | IDN-40% | IDN-50% |
| CausalNL | $65.33 \pm 0.64$ | $63.40 \pm 0.49$ | $55.37 \pm 1.09$ | $48.73 \pm 1.49$ | $38.69 \pm 1.72$ |
| CausalNL$'$ | $\mathbf{72.24 \pm 0.14}$ | $\mathbf{71.54 \pm 0.09}$ | $\mathbf{70.36 \pm 0.12}$ | $\mathbf{67.60 \pm 0.25}$ | $\mathbf{59.64 \pm 0.99}$ |
| InstanceGM | $79.30 \pm 0.11$ | $78.03 \pm 0.15$ | $77.89 \pm 0.18$ | $77.48 \pm 0.16$ | $76.64 \pm 0.38$ |
| InstanceGM$'$ | $\mathbf{79.87 \pm 0.09}$ | $\mathbf{79.53 \pm 0.17}$ | $\mathbf{79.43 \pm 0.16}$ | $\mathbf{79.45 \pm 0.15}$ | $\mathbf{79.23 \pm 0.12}$ |

size of 16. Both learning rates were reduced by a factor of 10 after 40 epochs. Due to limited space, more details about the hyperparameters are in Appendix D.

**Baselines** The baselines used in our experiments for comparison are: 1). CE, training the classification network using standard cross-entropy loss on noisy data directly; 2). MentorNet [22], pretraining a classification network to select reliable examples for the main classification network; 3). Co-Teaching [13], which uses two classification networks to select reliable examples for each other; 4). Reweight [36], using the importance reweighting method to estimate the unbiased risk defined on clean data; 5). Forward [40], which assumes the noise transition matrix is class-dependent, then corrects the loss function; 6). PTD [49], estimating instance-dependent noisy transition through the parts of instances; 7). CausalNL [55], which explores the information in the instances to help the learning of classification network; 8). CCR [9] uses forward-backward cycle-consistency regularization to learn noise transition matrtices; 9). MEIDTM [8], which uses Lipschitz continuity to constrain the noise transition matrix in the manifold; 10). BLTM [54], which learn the noise transition matrix on a part of dataset with Bayes optimal label; 11). NPC [1], which boosts the pre-trained classifier performances by calibrating the noisy predictions; 12). RENT [2], which utilizes the transition matrix for resampling. 13). DivideMix [28], which divides the noisy examples into labeled examples and unlabeled examples and trains the classification network using the semi-supervised technique MixMatch [4]; 14). SOP [35], which employs the sparse property of the label noise to prevent models from overfitting to label noise.

## 4.2 Effectiveness the Proposed Data-Generating Process

To demonstrate our data-generating process (shown in Fig. 2b) is more effective than the existing one (shown in Fig. 2a), we replace the generative model in CausalNL and InstanceGM with our proposed model while other experiment settings remain the same. We refer to the modified versions of CausalNL and InstanceGM as CausalNL$'$ and InstanceGM$'$, respectively. Experiments are conducted on CIFAR-10 and CIFAR-100 datasets with instance-dependent label noise. The experiment results are presented in Tab. 1 and 2, which show that CausalNL$'$ and InstanceGM$'$ outperform their original

Table 3: Means and standard deviations (percentage) of classification accuracy on Fashion-MNIST.

| | IDN-10% | IDN-20% | IDN-30% | IDN-40% | IDN-50% |
|---|---|---|---|---|---|
| | | | Fashion-MNIST | | |
| CE | 94.23 ± 0.08 | 93.94 ± 0.13 | 93.14 ± 0.03 | 89.54 ± 0.32 | 63.82 ± 3.87 |
| MentorNet | 93.16 ± 0.01 | 91.57 ± 0.29 | 90.52 ± 0.41 | 88.14 ± 0.76 | 61.62 ± 1.42 |
| Co-Teaching | 94.67 ± 0.08 | 94.23 ± 0.01 | 93.79 ± 0.07 | 92.83 ± 0.10 | 83.99 ± 4.57 |
| Reweight | 93.42 ± 0.16 | 93.12 ± 0.18 | 92.19 ± 0.18 | 88.51 ± 1.52 | 75.00 ± 5.28 |
| Forward | 93.48 ± 0.11 | 92.82 ± 0.12 | 91.05 ± 1.44 | 87.82 ± 1.81 | 78.34 ± 2.98 |
| PTD | 92.01 ± 0.35 | 91.08 ± 0.46 | 89.66 ± 0.43 | 85.69 ± 0.77 | 75.96 ± 1.38 |
| CausalNL | 93.97 ± 0.34 | 93.83 ± 0.14 | 93.37 ± 0.16 | 92.23 ± 0.24 | 90.13 ± 0.38 |
| CCR | 88.48 ± 0.16 | 83.59 ± 0.25 | 75.40 ± 0.19 | 64.39 ± 0.30 | 50.17 ± 0.29 |
| MEIDTM | 86.00 ± 0.84 | 80.99 ± 0.64 | 73.12 ± 2.34 | 63.81 ± 3.02 | 58.60 ± 3.32 |
| BLTM | 91.28 ± 1.93 | 91.20 ± 0.27 | 85.51 ± 4.77 | 82.42 ± 1.51 | 67.65 ± 5.65 |
| NPC | 88.78 ± 0.30 | 88.05 ± 0.02 | 84.99 ± 1.20 | 82.59 ± 1.22 | 70.58 ± 4.43 |
| RENT | 85.50 ± 0.57 | 79.82 ± 0.40 | 71.81 ± 0.86 | 61.70 ± 0.50 | 48.98 ± 1.57 |
| DivideMix | 95.24 ± 0.11 | 94.95 ± 0.10 | 94.36 ± 0.10 | 89.95 ± 0.15 | 83.35 ± 0.45 |
| SOP | **95.72 ± 0.08** | 95.40 ± 0.03 | 94.88 ± 0.03 | 92.54 ± 0.09 | 81.75 ± 0.30 |
| CSGN | 95.71 ± 0.10 | **95.46 ± 0.06** | **95.41 ± 0.07** | **95.25 ± 0.03** | **94.59 ± 0.05** |

Table 4: Means and standard deviations (percentage) of classification accuracy on CIFAR-10.

| | IDN-10% | IDN-20% | IDN-30% | IDN-40% | IDN-50% |
|---|---|---|---|---|---|
| | | | CIFAR-10 | | |
| CE | 88.22 ± 0.22 | 86.04 ± 0.38 | 82.40 ± 0.50 | 77.89 ± 1.46 | 53.97 ± 6.54 |
| MentorNet | 89.56 ± 0.19 | 88.80 ± 0.14 | 86.10 ± 0.12 | 81.13 ± 0.05 | 58.57 ± 0.71 |
| Co-Teaching | 90.32 ± 0.19 | 89.45 ± 0.33 | 87.08 ± 0.36 | 81.81 ± 0.62 | 52.09 ± 2.00 |
| Reweight | 89.63 ± 0.27 | 87.85 ± 0.97 | 81.29 ± 6.49 | 80.33 ± 3.75 | 75.14 ± 2.40 |
| Forward | 88.89 ± 0.18 | 87.83 ± 0.30 | 82.01 ± 3.29 | 79.49 ± 1.85 | 71.11 ± 8.78 |
| PTD | 79.01 ± 0.20 | 76.05 ± 0.53 | 72.28 ± 0.49 | 58.62 ± 0.88 | 53.98 ± 2.34 |
| CausalNL | 90.31 ± 0.09 | 89.76 ± 0.08 | 88.06 ± 0.53 | 80.97 ± 1.99 | 57.93 ± 7.53 |
| CCR | 91.43 ± 0.05 | 90.93 ± 0.07 | 90.15 ± 0.11 | 89.01 ± 0.15 | 86.05 ± 0.18 |
| MEIDTM | 86.52 ± 0.38 | 82.93 ± 0.44 | 77.35 ± 0.21 | 68.21 ± 2.09 | 57.84 ± 3.51 |
| BLTM | 80.16 ± 0.37 | 77.50 ± 1.30 | 71.47 ± 2.33 | 63.20 ± 4.52 | 48.12 ± 9.03 |
| NPC | 84.83 ± 0.22 | 83.13 ± 0.28 | 79.48 ± 0.31 | 73.85 ± 0.41 | 67.04 ± 0.06 |
| RENT | 80.90 ± 1.15 | 75.91 ± 1.39 | 74.06 ± 1.30 | 66.95 ± 2.82 | 52.73 ± 4.92 |
| DivideMix | 96.11 ± 0.07 | 95.75 ± 0.04 | 95.55 ± 0.23 | 95.30 ± 0.11 | 86.30 ± 0.14 |
| SOP | 96.08 ± 0.06 | 95.61 ± 0.05 | 94.14 ± 0.07 | 85.43 ± 0.24 | 63.43 ± 0.55 |
| CSGN | **96.46 ± 0.06** | **96.32 ± 0.05** | **96.27 ± 0.11** | **96.30 ± 0.11** | **95.88 ± 0.08** |

counterparts. This improvement indicates that our data-generating process is more effective than their data-generating process.

## 4.3 Noise Transition Matrices Estimation Error

We compare the noise transition matrices estimation error of the proposed method with other methods on the four datasets MNIST [27], Fashion-MNIST [51], CIFAR-10 [26] and CIFAR-100 [26]. The labels are corrupted manually by using the instance-dependent noisy label generation method proposed in [49]. The noise rates are from 0.1 to 0.5. The baselines used to compare the noise transition matrices estimation error are BLTM [54] and MEIDTM [8].

To calculate the noise transition matrix, we first estimate the clean class posterior $p(Y|X)$ using a classification network trained in the clean data. Then, the trained encoder and decoders are used to learn the causal variables, reconstruct images and noisy labels based on the instance $X$ and the sampling from the estimated $p(Y|X)$. The class posterior outputted by the noisy label decoder $f_{\tilde{Y}}$ is the estimated noise transition matrices. To quantify the estimation error, our methodology aligns with the precedent set by previous research [50], employing the relative error as the evaluative metric. The results of these experiments are shown in Fig. 5, which illustrates that our proposed method surpasses the other approaches in the estimation of noise transition matrices.

## 4.4 Classification Accuracy

We conduct experiments on both synthetic datasets [51, 26] and real-world datasets [48, 29]. To synthesize noisy labels, we employed instance-dependent noisy label generation methods for the synthetic datasets, as proposed by [49]. We experimented with noise rates of 0.1, 0.2, 0.3, 0.4, and 0.5, denoted by IDN-0.1, IDN-0.2, IDN-0.3, IDN-0.4, and IDN-0.5 respectively.

Table 5: Means and standard deviations (percentage) of classification accuracy on and CIFAR-100.

| | CIFAR-100 | | | | |
|---|---|---|---|---|---|
| | IDN-10% | IDN-20% | IDN-30% | IDN-40% | IDN-50% |
| CE | $63.46 \pm 0.56$ | $58.92 \pm 0.40$ | $52.55 \pm 0.18$ | $45.28 \pm 0.50$ | $37.48 \pm 0.62$ |
| MentorNet | $67.13 \pm 0.03$ | $65.83 \pm 0.18$ | $61.79 \pm 0.19$ | $55.57 \pm 0.37$ | $47.78 \pm 0.18$ |
| Co-Teaching | $66.44 \pm 0.41$ | $63.76 \pm 0.56$ | $58.37 \pm 0.27$ | $49.72 \pm 0.71$ | $37.77 \pm 1.31$ |
| Reweight | $59.38 \pm 0.33$ | $55.14 \pm 0.07$ | $46.91 \pm 1.26$ | $37.80 \pm 1.01$ | $28.45 \pm 2.57$ |
| Forward | $59.58 \pm 0.42$ | $56.59 \pm 0.25$ | $52.75 \pm 0.25$ | $46.03 \pm 0.65$ | $35.07 \pm 0.91$ |
| PTD | $67.33 \pm 0.33$ | $65.33 \pm 0.59$ | $64.56 \pm 1.55$ | $59.73 \pm 0.76$ | $56.80 \pm 1.32$ |
| CausalNL | $65.33 \pm 0.64$ | $63.40 \pm 0.49$ | $55.37 \pm 1.09$ | $48.73 \pm 1.49$ | $38.69 \pm 1.72$ |
| CCR | $69.73 \pm 0.07$ | $68.84 \pm 0.09$ | $67.65 \pm 0.08$ | $66.54 \pm 0.09$ | $64.66 \pm 0.11$ |
| MEIDTM | $69.88 \pm 0.45$ | $69.16 \pm 0.16$ | $66.76 \pm 0.30$ | $63.46 \pm 0.48$ | $59.18 \pm 0.16$ |
| BLTM | $48.82 \pm 0.44$ | $46.61 \pm 1.10$ | $41.35 \pm 0.85$ | $35.67 \pm 1.97$ | $29.28 \pm 0.74$ |
| NPC | $57.84 \pm 0.82$ | $53.42 \pm 0.31$ | $50.05 \pm 0.06$ | $46.53 \pm 0.07$ | $32.60 \pm 6.47$ |
| RENT | $38.88 \pm 2.24$ | $35.64 \pm 0.60$ | $31.53 \pm 1.03$ | $26.03 \pm 0.89$ | $21.11 \pm 0.60$ |
| DivideMix | $77.26 \pm 0.24$ | $76.81 \pm 0.14$ | $76.54 \pm 0.14$ | $73.12 \pm 0.32$ | $61.54 \pm 0.34$ |
| SOP | $78.68 \pm 0.06$ | $77.26 \pm 0.17$ | $74.94 \pm 0.06$ | $68.94 \pm 0.18$ | $60.03 \pm 0.12$ |
| CSGN | $\mathbf{80.59 \pm 0.07}$ | $\mathbf{79.35 \pm 0.12}$ | $\mathbf{78.91 \pm 0.11}$ | $\mathbf{78.58 \pm 0.15}$ | $\mathbf{74.60 \pm 0.17}$ |

Table 6: Means and standard deviations (percentage) of classification accuracy on CIFAR-N.

| | CIFAR-10N | | | | | CIFAR-100N |
|---|---|---|---|---|---|---|
| | Worst | Aggregate | Random 1 | Random 2 | Random 3 | Noisy |
| CE | $77.17 \pm 0.56$ | $88.28 \pm 0.20$ | $85.79 \pm 0.32$ | $85.35 \pm 0.44$ | $85.22 \pm 0.03$ | $52.45 \pm 0.38$ |
| MentorNet | $81.82 \pm 0.21$ | $89.92 \pm 0.10$ | $88.80 \pm 0.18$ | $88.68 \pm 0.15$ | $88.44 \pm 0.35$ | $57.70 \pm 0.09$ |
| Co-Teaching | $79.51 \pm 0.72$ | $90.59 \pm 0.19$ | $88.77 \pm 0.19$ | $89.00 \pm 0.38$ | $88.55 \pm 0.33$ | $55.11 \pm 1.18$ |
| Reweight | $77.68 \pm 2.46$ | $89.34 \pm 0.09$ | $88.44 \pm 0.10$ | $88.16 \pm 0.10$ | $88.03 \pm 0.10$ | $53.08 \pm 0.61$ |
| Forward | $79.27 \pm 1.18$ | $89.22 \pm 0.21$ | $86.84 \pm 0.97$ | $86.99 \pm 0.10$ | $87.53 \pm 0.34$ | $53.02 \pm 0.52$ |
| PTD | $65.62 \pm 5.28$ | $84.66 \pm 3.28$ | $82.11 \pm 3.17$ | $74.76 \pm 9.98$ | $84.29 \pm 0.64$ | $23.68 \pm 2.65$ |
| CausalNL | $78.48 \pm 2.35$ | $90.24 \pm 0.12$ | $88.08 \pm 0.38$ | $87.54 \pm 0.34$ | $87.67 \pm 0.16$ | $53.38 \pm 1.42$ |
| CCR | $80.43 \pm 0.24$ | $90.10 \pm 0.09$ | $88.53 \pm 0.08$ | $88.21 \pm 0.11$ | $88.46 \pm 0.08$ | $57.21 \pm 0.25$ |
| MEIDTM | $79.59 \pm 0.89$ | $90.15 \pm 0.27$ | $87.81 \pm 0.52$ | $88.07 \pm 0.18$ | $87.86 \pm 0.21$ | $38.90 \pm 0.91$ |
| BLTM | $68.21 \pm 1.67$ | $79.41 \pm 1.00$ | $78.09 \pm 1.03$ | $76.99 \pm 1.23$ | $76.26 \pm 0.71$ | $41.86 \pm 0.46$ |
| NPC | $75.40 \pm 0.48$ | $85.56 \pm 0.47$ | $83.07 \pm 0.30$ | $83.34 \pm 0.18$ | $83.27 \pm 0.36$ | $46.02 \pm 0.36$ |
| RENT | $70.01 \pm 2.23$ | $80.94 \pm 0.53$ | $77.46 \pm 0.99$ | $77.74 \pm 1.06$ | $77.42 \pm 0.15$ | $32.79 \pm 1.39$ |
| DivideMix | $93.41 \pm 0.19$ | $95.12 \pm 0.15$ | $95.32 \pm 0.13$ | $95.15 \pm 0.09$ | $95.23 \pm 0.16$ | $69.45 \pm 0.19$ |
| SOP | $93.24 \pm 0.21$ | $95.61 \pm 0.13$ | $95.28 \pm 0.13$ | $95.31 \pm 0.10$ | $95.39 \pm 0.11$ | $67.81 \pm 0.23$ |
| CSGN | $\mathbf{94.01 \pm 0.12}$ | $\mathbf{95.87 \pm 0.08}$ | $\mathbf{95.47 \pm 0.12}$ | $\mathbf{95.45 \pm 0.09}$ | $\mathbf{95.54 \pm 0.06}$ | $\mathbf{71.99 \pm 0.08}$ |

The experimental results for synthetic datasets are presented in Tab. 3, Tab. 4 and Tab. 5. The real-world dataset experiment results are presented in Tab. 6 and Tab. 7. Generally, our proposed method outperforms existing methods in terms of test accuracy on both synthetic and real-world datasets containing label noise. In particular, when the noise rate is equal to 50%, the accuracy of the model is about ten percentage points higher than the best baseline method. The results demonstrate that the proposed method can capture the noise transition matrix under different settings and improve the classification performance.

Table 7: Test accuracy of CSGN on the WebVision validation set and the ImageNet ILSVRC12 validation set.

| | WebVision | | ILSVRC12 | |
|---|---|---|---|---|
| | top-1 | top-5 | top-1 | top-5 |
| Forward | 61.12 | 82.68 | 57.36 | 82.36 |
| Decoupling | 62.54 | 84.74 | 58.26 | 82.26 |
| MentorNet | 63.00 | 81.40 | 57.80 | 79.92 |
| Co-Teaching | 63.58 | 85.20 | 61.48 | 84.70 |
| DivideMix | 77.32 | 91.64 | 75.20 | 90.84 |
| ELR+ | 77.78 | 91.68 | 70.29 | 89.76 |
| CSGN | $\mathbf{79.84}$ | $\mathbf{93.52}$ | $\mathbf{76.56}$ | $\mathbf{93.76}$ |

## 5 Conclusion

Noise transition matrices are important for many label-noise learning algorithms. However, current label-noise learning methods often can only estimate the noise transition matrices of some instances. To leverage these estimated transition matrices to help estimate transition matrices of other instances. It is crucial to establish the connection among the noise transition matrices for different instances. Prior work tackled this issue by manually defining the similarities of noise transition matrices across different instances. Given only noisy data, the introduced similarity-based assumptions are hard to verify. If similarities are not truthful, the estimation error of the noise transition matrix could be large, leading to performance degeneration for label-noise learning algorithms. We propose an effective method to capture the connection among the noise transition matrices implicitly by modeling the latent causal structures in the generation of noisy data. Experimental results on different noisy datasets show that our method achieves state-of-the-art performance in estimating noise transition matrices, which leads to improved classification accuracy.

## Acknowledgments and Disclosure of Funding

Tongliang Liu is partially supported by the following Australian Research Council projects: FT220100318, DP220102121, LP220100527, LP220200949, and IC190100031.

## References

[1] HeeSun Bae, Seungjae Shin, Byeonghu Na, JoonHo Jang, Kyungwoo Song, and Il-Chul Moon. From noisy prediction to true label: Noisy prediction calibration via generative model. In Kamalika Chaudhuri, Stefanie Jegelka, Le Song, Csaba Szepesvári, Gang Niu, and Sivan Sabato, editors, *International Conference on Machine Learning, ICML 2022, 17-23 July 2022, Baltimore, Maryland, USA*, volume 162 of *Proceedings of Machine Learning Research*, pages 1277–1297. PMLR, 2022.

[2] HeeSun Bae, Seungjae Shin, Byeonghu Na, and Il-Chul Moon. Dirichlet-based per-sample weighting by transition matrix for noisy label learning. *arXiv preprint arXiv:2403.02690*, 2024.

[3] Yingbin Bai, Erkun Yang, Bo Han, Yanhua Yang, Jiatong Li, Yinian Mao, Gang Niu, and Tongliang Liu. Understanding and improving early stopping for learning with noisy labels. In Marc'Aurelio Ranzato, Alina Beygelzimer, Yann N. Dauphin, Percy Liang, and Jennifer Wortman Vaughan, editors, *Advances in Neural Information Processing Systems 34: Annual Conference on Neural Information Processing Systems 2021, NeurIPS 2021, December 6-14, 2021, virtual*, pages 24392–24403, 2021.

[4] David Berthelot, Nicholas Carlini, Ian J. Goodfellow, Nicolas Papernot, Avital Oliver, and Colin Raffel. Mixmatch: A holistic approach to semi-supervised learning. In Hanna M. Wallach, Hugo Larochelle, Alina Beygelzimer, Florence d'Alché-Buc, Emily B. Fox, and Roman Garnett, editors, *Advances in Neural Information Processing Systems 32: Annual Conference on Neural Information Processing Systems 2019, NeurIPS 2019, December 8-14, 2019, Vancouver, BC, Canada*, pages 5050–5060, 2019.

[5] Avrim Blum, Adam Kalai, and Hal Wasserman. Noise-tolerant learning, the parity problem, and the statistical query model. *J. ACM*, 50(4):506–519, 2003.

[6] Johann Brehmer, Pim de Haan, Phillip Lippe, and Taco S. Cohen. Weakly supervised causal representation learning. In *NeurIPS*, 2022.

[7] Pengfei Chen, Ben Ben Liao, Guangyong Chen, and Shengyu Zhang. Understanding and utilizing deep neural networks trained with noisy labels. In *International Conference on Machine Learning*, pages 1062–1070. PMLR, 2019.

[8] De Cheng, Tongliang Liu, Yixiong Ning, Nannan Wang, Bo Han, Gang Niu, Xinbo Gao, and Masashi Sugiyama. Instance-dependent label-noise learning with manifold-regularized transition matrix estimation. In *IEEE/CVF Conference on Computer Vision and Pattern Recognition, CVPR 2022, New Orleans, LA, USA, June 18-24, 2022*, pages 16609–16618. IEEE, 2022.

[9] De Cheng, Yixiong Ning, Nannan Wang, Xinbo Gao, Heng Yang, Yuxuan Du, Bo Han, and Tongliang Liu. Class-dependent label-noise learning with cycle-consistency regularization. In *NeurIPS*, 2022.

[10] Arpit Garg, Cuong Nguyen, Rafael Felix, Thanh-Toan Do, and Gustavo Carneiro. Instance-dependent noisy label learning via graphical modelling. In *Proceedings of the IEEE/CVF Winter Conference on Applications of Computer Vision (WACV)*, pages 2288–2298, January 2023.

[11] Aritra Ghosh, Himanshu Kumar, and P Shanti Sastry. Robust loss functions under label noise for deep neural networks. In *Proceedings of the AAAI conference on artificial intelligence*, volume 31, 2017.

[12] Aritra Ghosh, Naresh Manwani, and PS Sastry. Making risk minimization tolerant to label noise. *Neurocomputing*, 160:93–107, 2015.

[13] Bo Han, Quanming Yao, Xingrui Yu, Gang Niu, Miao Xu, Weihua Hu, Ivor W. Tsang, and Masashi Sugiyama. Co-teaching: Robust training of deep neural networks with extremely noisy labels. In Samy Bengio, Hanna M. Wallach, Hugo Larochelle, Kristen Grauman, Nicolò Cesa-Bianchi, and Roman Garnett, editors, *Advances in Neural Information Processing Systems 31: Annual Conference on Neural Information Processing Systems 2018, NeurIPS 2018, December 3-8, 2018, Montréal, Canada*, pages 8536–8546, 2018.

[14] Kaiming He, Xiangyu Zhang, Shaoqing Ren, and Jian Sun. Identity mappings in deep residual networks. In Bastian Leibe, Jiri Matas, Nicu Sebe, and Max Welling, editors, *Computer Vision - ECCV 2016 - 14th European Conference, Amsterdam, The Netherlands, October 11-14, 2016, Proceedings, Part IV*, volume 9908 of *Lecture Notes in Computer Science*, pages 630–645. Springer, 2016.

[15] Zhuo Huang, Chang Liu, Yinpeng Dong, Hang Su, Shibao Zheng, and Tongliang Liu. Machine vision therapy: Multimodal large language models can enhance visual robustness via denoising in-context learning. In *Forty-first International Conference on Machine Learning*, 2023.

[16] Zhuo Huang, Li Shen, Jun Yu, Bo Han, and Tongliang Liu. Flatmatch: Bridging labeled data and unlabeled data with cross-sharpness for semi-supervised learning. *Advances in Neural Information Processing Systems*, 36:18474–18494, 2023.

[17] Zhuo Huang, Xiaobo Xia, Li Shen, Bo Han, Mingming Gong, Chen Gong, and Tongliang Liu. Harnessing out-of-distribution examples via augmenting content and style. In *The Eleventh International Conference on Learning Representations*, 2023.

[18] Aapo Hyvärinen and Hiroshi Morioka. Unsupervised feature extraction by time-contrastive learning and nonlinear ICA. In Daniel D. Lee, Masashi Sugiyama, Ulrike von Luxburg, Isabelle Guyon, and Roman Garnett, editors, *Advances in Neural Information Processing Systems 29: Annual Conference on Neural Information Processing Systems 2016, December 5-10, 2016, Barcelona, Spain*, pages 3765–3773, 2016.

[19] Aapo Hyvärinen and Hiroshi Morioka. Nonlinear ICA of temporally dependent stationary sources. In Aarti Singh and Xiaojin (Jerry) Zhu, editors, *Proceedings of the 20th International Conference on Artificial Intelligence and Statistics, AISTATS 2017, 20-22 April 2017, Fort Lauderdale, FL, USA*, volume 54 of *Proceedings of Machine Learning Research*, pages 460–469. PMLR, 2017.

[20] Aapo Hyvärinen and Petteri Pajunen. Nonlinear independent component analysis: Existence and uniqueness results. *Neural Networks*, 12(3):429–439, 1999.

[21] Aapo Hyvärinen, Hiroaki Sasaki, and Richard E. Turner. Nonlinear ICA using auxiliary variables and generalized contrastive learning. In Kamalika Chaudhuri and Masashi Sugiyama, editors, *The 22nd International Conference on Artificial Intelligence and Statistics, AISTATS 2019, 16-18 April 2019, Naha, Okinawa, Japan*, volume 89 of *Proceedings of Machine Learning Research*, pages 859–868. PMLR, 2019.

[22] Lu Jiang, Zhengyuan Zhou, Thomas Leung, Li-Jia Li, and Li Fei-Fei. Mentornet: Learning data-driven curriculum for very deep neural networks on corrupted labels. In Jennifer G. Dy and Andreas Krause, editors, *Proceedings of the 35th International Conference on Machine Learning, ICML 2018, Stockholmsmässan, Stockholm, Sweden, July 10-15, 2018*, volume 80 of *Proceedings of Machine Learning Research*, pages 2309–2318. PMLR, 2018.

[23] Ilyes Khemakhem, Diederik P. Kingma, Ricardo Pio Monti, and Aapo Hyvärinen. Variational autoencoders and nonlinear ICA: A unifying framework. In Silvia Chiappa and Roberto Calandra, editors, *The 23rd International Conference on Artificial Intelligence and Statistics, AISTATS 2020, 26-28 August 2020, Online [Palermo, Sicily, Italy]*, volume 108 of *Proceedings of Machine Learning Research*, pages 2207–2217. PMLR, 2020.

[24] Diederik P Kingma and Max Welling. Auto-encoding variational bayes. *arXiv preprint arXiv:1312.6114*, 2013.

[25] Daphne Koller and Nir Friedman. *Probabilistic graphical models: principles and techniques*. MIT press, 2009.

[26] Alex Krizhevsky, Geoffrey Hinton, et al. Learning multiple layers of features from tiny images. 2009.

[27] Yann LeCun, Corinna Cortes, and CJ Burges. Mnist handwritten digit database. *ATT Labs [Online]. Available: http://yann. lecun. com/exdb/mnist*, 2, 2010.

[28] Junnan Li, Richard Socher, and Steven C. H. Hoi. Dividemix: Learning with noisy labels as semi-supervised learning. In *8th International Conference on Learning Representations, ICLR 2020, Addis Ababa, Ethiopia, April 26-30, 2020*. OpenReview.net, 2020.

[29] Wen Li, Limin Wang, Wei Li, Eirikur Agustsson, and Luc Van Gool. Webvision database: Visual learning and understanding from web data. *arXiv preprint arXiv:1708.02862*, 2017.

[30] Xuefeng Li, Tongliang Liu, Bo Han, Gang Niu, and Masashi Sugiyama. Provably end-to-end label-noise learning without anchor points. In Marina Meila and Tong Zhang, editors, *Proceedings of the 38th International Conference on Machine Learning, ICML 2021, 18-24 July 2021, Virtual Event*, volume 139 of *Proceedings of Machine Learning Research*, pages 6403–6413. PMLR, 2021.

[31] Yuncheng Li, Jianchao Yang, Yale Song, Liangliang Cao, Jiebo Luo, and Li-Jia Li. Learning from noisy labels with distillation. In *IEEE International Conference on Computer Vision, ICCV 2017, Venice, Italy, October 22-29, 2017*, pages 1928–1936. IEEE Computer Society, 2017.

[32] Yexiong Lin, Yu Yao, Yuxuan Du, Jun Yu, Bo Han, Mingming Gong, and Tongliang Liu. Do we need to penalize variance of losses for learning with label noise? *arXiv preprint arXiv:2201.12739*, 2022.

[33] Yexiong Lin, Yu Yao, Xiaolong Shi, Mingming Gong, Xu Shen, Dong Xu, and Tongliang Liu. Cs-isolate: Extracting hard confident examples by content and style isolation. *Advances in Neural Information Processing Systems*, 36, 2024.

[34] Sheng Liu, Jonathan Niles-Weed, Narges Razavian, and Carlos Fernandez-Granda. Early-learning regularization prevents memorization of noisy labels. In Hugo Larochelle, Marc'Aurelio Ranzato, Raia Hadsell, Maria-Florina Balcan, and Hsuan-Tien Lin, editors, *Advances in Neural Information Processing Systems 33: Annual Conference on Neural Information Processing Systems 2020, NeurIPS 2020, December 6-12, 2020, virtual*, 2020.

[35] Sheng Liu, Zhihui Zhu, Qing Qu, and Chong You. Robust training under label noise by over-parameterization. In Kamalika Chaudhuri, Stefanie Jegelka, Le Song, Csaba Szepesvári, Gang Niu, and Sivan Sabato, editors, *International Conference on Machine Learning, ICML 2022, 17-23 July 2022, Baltimore, Maryland, USA*, volume 162 of *Proceedings of Machine Learning Research*, pages 14153–14172. PMLR, 2022.

[36] Tongliang Liu and Dacheng Tao. Classification with noisy labels by importance reweighting. *IEEE Trans. Pattern Anal. Mach. Intell.*, 38(3):447–461, 2016.

[37] Yang Liu, Hao Cheng, and Kun Zhang. Identifiability of label noise transition matrix. In *International Conference on Machine Learning*, pages 21475–21496. PMLR, 2023.

[38] Yuhang Liu, Zhen Zhang, Dong Gong, Mingming Gong, Biwei Huang, Anton van den Hengel, Kun Zhang, and Javen Qinfeng Shi. Identifying weight-variant latent causal models. *arXiv preprint arXiv:2208.14153*, 2022.

[39] Francesco Locatello, Stefan Bauer, Mario Lucic, Gunnar Rätsch, Sylvain Gelly, Bernhard Schölkopf, and Olivier Bachem. Challenging common assumptions in the unsupervised learning of disentangled representations. In Kamalika Chaudhuri and Ruslan Salakhutdinov, editors, *Proceedings of the 36th International Conference on Machine Learning, ICML 2019, 9-15 June 2019, Long Beach, California, USA*, volume 97 of *Proceedings of Machine Learning Research*, pages 4114–4124. PMLR, 2019.

[40] Giorgio Patrini, Alessandro Rozza, Aditya Krishna Menon, Richard Nock, and Lizhen Qu. Making deep neural networks robust to label noise: A loss correction approach. In *2017 IEEE Conference on Computer Vision and Pattern Recognition, CVPR 2017, Honolulu, HI, USA, July 21-26, 2017*, pages 2233–2241. IEEE Computer Society, 2017.

[41] Hans Reichenbach. *The Direction of Time*. Dover Publications, Mineola, N.Y., 1956.

[42] Bernhard Schölkopf. Causality for machine learning. In *Probabilistic and causal inference: The works of Judea Pearl*, pages 765–804. 2022.

[43] Bernhard Schölkopf, Francesco Locatello, Stefan Bauer, Nan Rosemary Ke, Nal Kalchbrenner, Anirudh Goyal, and Yoshua Bengio. Toward causal representation learning. *Proc. IEEE*, 109(5):612–634, 2021.

[44] Henning Sprekeler, Tiziano Zito, and Laurenz Wiskott. An extension of slow feature analysis for nonlinear blind source separation. *J. Mach. Learn. Res.*, 15(1):921–947, 2014.

[45] Christian Szegedy, Sergey Ioffe, Vincent Vanhoucke, and Alexander Alemi. Inception-v4, inception-resnet and the impact of residual connections on learning. In *Proceedings of the AAAI conference on artificial intelligence*, volume 31, 2017.

[46] Daiki Tanaka, Daiki Ikami, Toshihiko Yamasaki, and Kiyoharu Aizawa. Joint optimization framework for learning with noisy labels. In *Proceedings of the IEEE conference on computer vision and pattern recognition*, pages 5552–5560, 2018.

[47] Yisen Wang, Xingjun Ma, Zaiyi Chen, Yuan Luo, Jinfeng Yi, and James Bailey. Symmetric cross entropy for robust learning with noisy labels. In *Proceedings of the IEEE/CVF international conference on computer vision*, pages 322–330, 2019.

[48] Jiaheng Wei, Zhaowei Zhu, Hao Cheng, Tongliang Liu, Gang Niu, and Yang Liu. Learning with noisy labels revisited: A study using real-world human annotations. In *The Tenth International Conference on Learning Representations, ICLR 2022, Virtual Event, April 25-29, 2022*. OpenReview.net, 2022.

[49] Xiaobo Xia, Tongliang Liu, Bo Han, Nannan Wang, Mingming Gong, Haifeng Liu, Gang Niu, Dacheng Tao, and Masashi Sugiyama. Part-dependent label noise: Towards instance-dependent label noise. In Hugo Larochelle, Marc'Aurelio Ranzato, Raia Hadsell, Maria-Florina Balcan, and Hsuan-Tien Lin, editors, *Advances in Neural Information Processing Systems 33: Annual Conference on Neural Information Processing Systems 2020, NeurIPS 2020, December 6-12, 2020, virtual*, 2020.

[50] Xiaobo Xia, Tongliang Liu, Nannan Wang, Bo Han, Chen Gong, Gang Niu, and Masashi Sugiyama. Are anchor points really indispensable in label-noise learning? In Hanna M. Wallach, Hugo Larochelle, Alina Beygelzimer, Florence d'Alché-Buc, Emily B. Fox, and Roman Garnett, editors, *Advances in Neural Information Processing Systems 32: Annual Conference on Neural Information Processing Systems 2019, NeurIPS 2019, December 8-14, 2019, Vancouver, BC, Canada*, pages 6835–6846, 2019.

[51] Han Xiao, Kashif Rasul, and Roland Vollgraf. Fashion-mnist: a novel image dataset for benchmarking machine learning algorithms. *CoRR*, abs/1708.07747, 2017.

[52] Yan Yan, Rómer Rosales, Glenn Fung, Subramanian Ramanathan, and Jennifer G. Dy. Learning from multiple annotators with varying expertise. *Mach. Learn.*, 95(3):291–327, 2014.

[53] Mengyue Yang, Furui Liu, Zhitang Chen, Xinwei Shen, Jianye Hao, and Jun Wang. Causalvae: Disentangled representation learning via neural structural causal models. In *IEEE Conference on Computer Vision and Pattern Recognition, CVPR 2021, virtual, June 19-25, 2021*, pages 9593–9602. Computer Vision Foundation / IEEE, 2021.

[54] Shuo Yang, Erkun Yang, Bo Han, Yang Liu, Min Xu, Gang Niu, and Tongliang Liu. Estimating instance-dependent bayes-label transition matrix using a deep neural network. In Kamalika Chaudhuri, Stefanie Jegelka, Le Song, Csaba Szepesvári, Gang Niu, and Sivan Sabato, editors, *International Conference on Machine Learning, ICML 2022, 17-23 July 2022, Baltimore, Maryland, USA*, volume 162 of *Proceedings of Machine Learning Research*, pages 25302–25312. PMLR, 2022.

[55] Yu Yao, Tongliang Liu, Mingming Gong, Bo Han, Gang Niu, and Kun Zhang. Instance-dependent label-noise learning under a structural causal model. In Marc'Aurelio Ranzato, Alina Beygelzimer, Yann N. Dauphin, Percy Liang, and Jennifer Wortman Vaughan, editors, *Advances in Neural Information Processing Systems 34: Annual Conference on Neural Information Processing Systems 2021, NeurIPS 2021, December 6-14, 2021, virtual*, pages 4409–4420, 2021.

[56] Yu Yao, Tongliang Liu, Bo Han, Mingming Gong, Jiankang Deng, Gang Niu, and Masashi Sugiyama. Dual t: Reducing estimation error for transition matrix in label-noise learning. *Advances in neural information processing systems*, 33:7260–7271, 2020.

[57] Suqin Yuan, Lei Feng, and Tongliang Liu. Late stopping: Avoiding confidently learning from mislabeled examples. In *Proceedings of the IEEE/CVF International Conference on Computer Vision*, pages 16079–16088, 2023.

[58] Suqin Yuan, Lei Feng, and Tongliang Liu. Early stopping against label noise without validation data. In *The Twelfth International Conference on Learning Representations*, 2024.

[59] Chiyuan Zhang, Samy Bengio, Moritz Hardt, Benjamin Recht, and Oriol Vinyals. Understanding deep learning requires rethinking generalization. In *5th International Conference on Learning Representations, ICLR 2017, Toulon, France, April 24-26, 2017, Conference Track Proceedings*. OpenReview.net, 2017.

[60] Songzhu Zheng, Pengxiang Wu, Aman Goswami, Mayank Goswami, Dimitris Metaxas, and Chao Chen. Error-bounded correction of noisy labels. In *International Conference on Machine Learning*, pages 11447–11457. PMLR, 2020.

[61] Zhaowei Zhu, Yiwen Song, and Yang Liu. Clusterability as an alternative to anchor points when learning with noisy labels. In Marina Meila and Tong Zhang, editors, *Proceedings of the 38th International Conference on Machine Learning, ICML 2021, 18-24 July 2021, Virtual Event*, volume 139 of *Proceedings of Machine Learning Research*, pages 12912–12923. PMLR, 2021.

# Appendix

## Table of Contents

## A   Derivation of ELBO

The derivation of ELBO is shown as follows:

$$
\mathbb{E}_{(\boldsymbol{x},\tilde{y},y)\sim q_{\mathcal{D}}}[\log p_{\theta}(\boldsymbol{x},\tilde{y}|y)]
$$

$$
=\mathbb{E}_{(\boldsymbol{x},\tilde{y},y)\sim q_{\mathcal{D}}}\left[\log \frac{p_{\theta}(\boldsymbol{x},\tilde{y},\boldsymbol{z},\boldsymbol{s}|y)}{q_{\varphi}(\boldsymbol{z},\boldsymbol{s}|\boldsymbol{x},y)}\frac{q_{\varphi}(\boldsymbol{z},\boldsymbol{s}|\boldsymbol{x},y)}{p_{\theta}(\boldsymbol{z},\boldsymbol{s}|\boldsymbol{x},\tilde{y},y)}\right] \tag{7}
$$

$$
=\mathbb{E}_{(\boldsymbol{x},\tilde{y},y)\sim q_{\mathcal{D}}}\left[\iint q_{\varphi}(\boldsymbol{z},\boldsymbol{s}|\boldsymbol{x},y)\log \frac{p_{\theta}(\boldsymbol{x},\tilde{y},\boldsymbol{z},\boldsymbol{s}|y)}{q_{\varphi}(\boldsymbol{z},\boldsymbol{s}|\boldsymbol{x},y)}\frac{q_{\varphi}(\boldsymbol{z},\boldsymbol{s}|\boldsymbol{x},y)}{p_{\theta}(\boldsymbol{z},\boldsymbol{s}|\boldsymbol{x},\tilde{y},y)}d\boldsymbol{z}d\boldsymbol{s}\right] \tag{8}
$$

$$
=\mathbb{E}_{(\boldsymbol{x},\tilde{y},y)\sim q_{\mathcal{D}}}\left[\iint q_{\varphi}(\boldsymbol{z},\boldsymbol{s}|\boldsymbol{x},y)\log \frac{p_{\theta}(\boldsymbol{x},\tilde{y},\boldsymbol{z},\boldsymbol{s}|y)}{q_{\varphi}(\boldsymbol{z},\boldsymbol{s}|\boldsymbol{x},y)}d\boldsymbol{z}d\boldsymbol{s}\right.
$$
$$
\left.+KL(q_{\varphi}(\boldsymbol{z},\boldsymbol{s}|\boldsymbol{x},y)||p_{\theta}(\boldsymbol{z},\boldsymbol{s}|\boldsymbol{x},\tilde{y},y))\right] \tag{9}
$$

$$
\geq \mathbb{E}_{(\boldsymbol{x},\tilde{y},y)\sim q_{\mathcal{D}}}\left[\iint q_{\varphi}(\boldsymbol{z},\boldsymbol{s}|\boldsymbol{x},y)\log \frac{p_{\theta}(\boldsymbol{x},\tilde{y},\boldsymbol{z},\boldsymbol{s}|y)}{q_{\varphi}(\boldsymbol{z},\boldsymbol{s}|\boldsymbol{x},y)}d\boldsymbol{z}d\boldsymbol{s}\right]. \tag{10}
$$

$$ELBO = \mathbb{E}_{(\boldsymbol{x},\tilde{y},y)\sim q_{\mathcal{D}}}\left[\iint q_{\varphi}(\boldsymbol{z},\boldsymbol{s}|\boldsymbol{x},y)\log\frac{p_{\theta}(\boldsymbol{x},\tilde{y},\boldsymbol{z},\boldsymbol{s}|y)}{q_{\varphi}(\boldsymbol{z},\boldsymbol{s}|\boldsymbol{x},y)}d\boldsymbol{z}d\boldsymbol{s}\right] \tag{11}$$

$$= \mathbb{E}_{(\boldsymbol{x},\tilde{y},y)\sim q_{\mathcal{D}}}\left[\iint q_{\varphi}(\boldsymbol{z},\boldsymbol{s}|\boldsymbol{x},y)\log\frac{p_{\boldsymbol{f}}(\boldsymbol{x},\tilde{y}|\boldsymbol{z},\boldsymbol{s},y)p_{\boldsymbol{W},\boldsymbol{\mu_N},\boldsymbol{\sigma_N^2}}(\boldsymbol{z}|y)p(\boldsymbol{s})}{q_{\varphi}(\boldsymbol{z},\boldsymbol{s}|\boldsymbol{x},y)}d\boldsymbol{z}d\boldsymbol{s}\right] \tag{12}$$

$$= \mathbb{E}_{(\boldsymbol{x},\tilde{y},y)\sim q_{\mathcal{D}}}\left[\iint q_{\varphi}(\boldsymbol{z},\boldsymbol{s}|\boldsymbol{x},y)\log p_{\boldsymbol{f}}(\boldsymbol{x},\tilde{y}|\boldsymbol{z},\boldsymbol{s},y)d\boldsymbol{z}d\boldsymbol{s}\right.$$
$$\left. + \iint q_{\varphi}(\boldsymbol{z}|\boldsymbol{x},y)q_{\varphi}(\boldsymbol{s}|\boldsymbol{x},y)\log\frac{p_{\boldsymbol{W},\boldsymbol{\mu_N},\boldsymbol{\sigma_N^2}}(\boldsymbol{z}|y)p(\boldsymbol{s})}{q_{\varphi}(\boldsymbol{z}|\boldsymbol{x},y)q_{\varphi}(\boldsymbol{s}|\boldsymbol{x},y)}d\boldsymbol{z}d\boldsymbol{s}\right] \tag{13}$$

$$= \mathbb{E}_{(\boldsymbol{x},\tilde{y},y)\sim q_{\mathcal{D}}}\left[\mathbb{E}_{(\boldsymbol{z},\boldsymbol{s})\sim q_{\varphi}}[\log p_{\boldsymbol{f}}(\boldsymbol{x},\tilde{y}|\boldsymbol{z},\boldsymbol{s},y)]\right.$$
$$\left. - KL(q_{\varphi}(\boldsymbol{z}|\boldsymbol{x},y)||p_{\boldsymbol{W},\boldsymbol{\mu_N},\boldsymbol{\sigma_N^2}}(\boldsymbol{z}|y)) - KL(q_{\varphi}(\boldsymbol{s}|\boldsymbol{x},y)||p(\boldsymbol{s}))\right] \tag{14}$$

$$= \mathbb{E}_{(\boldsymbol{x},\tilde{y},y)\sim q_{\tilde{\mathcal{D}}}q_{\psi}}\left[\mathbb{E}_{(\boldsymbol{z},\boldsymbol{s})\sim q_{\varphi}}[\log p_{\boldsymbol{f}}(\boldsymbol{x},\tilde{y}|\boldsymbol{z},\boldsymbol{s})]\right.$$
$$\left. - KL(q_{\varphi}(\boldsymbol{z}|\boldsymbol{x},y)||p_{\boldsymbol{W},\boldsymbol{\mu_N},\boldsymbol{\sigma_N^2}}(\boldsymbol{z}|y)) - KL(q_{\varphi}(\boldsymbol{s}|\boldsymbol{x},y)||p(\boldsymbol{s}))\right], \tag{15}$$

where Eq. 13 holds because we the approximate the distribution $q_{\varphi}(\boldsymbol{Z},\boldsymbol{S}|\boldsymbol{X},Y)$ by the posterior distribution $q_{\varphi}(\boldsymbol{Z}|\boldsymbol{X},Y)$ and $q_{\varphi}(\boldsymbol{S}|\boldsymbol{X},Y)$, i.e., $q_{\varphi}(\boldsymbol{Z},\boldsymbol{S}|\boldsymbol{X},Y) \approx q_{\varphi}(\boldsymbol{Z}|\boldsymbol{X},Y)q_{\varphi}(\boldsymbol{S}|\boldsymbol{X},Y)$.

# B  Identifiability Analysis

In this section, we discuss the required conditions for identifying latent causal structures.

The data-generating process can be defined as:

$$p_{\boldsymbol{f}}(\boldsymbol{X},\tilde{Y}|\boldsymbol{Z},\boldsymbol{S}) = p_{\boldsymbol{f_X}}(\boldsymbol{X}|\boldsymbol{M_X}\odot\boldsymbol{Z},\boldsymbol{S})p_{\boldsymbol{f_{\tilde{Y}}}}(\tilde{Y}|\boldsymbol{M_{\tilde{Y}}}\odot\boldsymbol{Z})$$
$$= p_{\boldsymbol{\varepsilon_X}}(\boldsymbol{X} - p_{\boldsymbol{f_X}}(\boldsymbol{M_X}\odot\boldsymbol{Z},\boldsymbol{S}))p_{\boldsymbol{\varepsilon_{\tilde{Y}}}}(\tilde{Y} - p_{\boldsymbol{f_{\tilde{Y}}}}(\boldsymbol{M_{\tilde{Y}}}\odot\boldsymbol{Z})), \tag{16}$$

which means that the value of $\boldsymbol{X}$ and $\tilde{Y}$ can be decomposed as $\boldsymbol{X} = \boldsymbol{f_X}(\boldsymbol{M_X}\odot\boldsymbol{Z},\boldsymbol{S}) + \boldsymbol{\varepsilon_X}, \tilde{Y} = \boldsymbol{f_{\tilde{Y}}}(\boldsymbol{M_{\tilde{Y}}}\odot\boldsymbol{Z}) + \boldsymbol{\varepsilon_{\tilde{Y}}}$, where $\boldsymbol{\varepsilon_X}$ and $\boldsymbol{\varepsilon_{\tilde{Y}}}$ are independent noise variables with probability density functions $p_{\boldsymbol{\varepsilon_X}}(\boldsymbol{\varepsilon_X})$ and $p_{\boldsymbol{\varepsilon_{\tilde{Y}}}}(\boldsymbol{\varepsilon_{\tilde{Y}}})$. The variables $\boldsymbol{S}$ represent other variables unrelated to the clean label in the instances, such as the rotation and brightness [17].

Intuitively, the instances are generated by a subset of the causal variables and the variables $\boldsymbol{S}$, while the noisy labels are generated by another subset of the causal variables. Let $\boldsymbol{f^s}$ represent the combination of the functions $\boldsymbol{f_X}$ and $\boldsymbol{f_{\tilde{Y}}}$ given a style $\boldsymbol{s}$.

Let $m$ denote the number of the causal variables, which is also the number of the latent noise variables; Let $k$ denote the dimension of the sufficient statistics for the causal variables $\boldsymbol{Z}$. We have the following theorem:

**Theorem B.1.** *[38] Suppose latent causal variables $\boldsymbol{Z}$ and the observed variables $Y,\tilde{Y}$ follow the generative model defined in Eq. (4) with parameters $(\boldsymbol{f^s},\boldsymbol{W},\boldsymbol{\mu_N},\boldsymbol{\sigma_N^2})$. Given a style $\boldsymbol{s}$, assume the following holds:*

1. *The set $\{\boldsymbol{x}\in\mathcal{X}|\varphi_{\boldsymbol{\varepsilon}}(\boldsymbol{x})\}$ has measure zero, where is the characteristic function of the density $p_{\boldsymbol{\varepsilon}}$.*

2. *The function $\boldsymbol{f^s}$ is bijective.*

3. *There exist $2m+1$ distinct points $\boldsymbol{y_{N_0}},\boldsymbol{y_{N_1}},\ldots,\boldsymbol{y_{N_{2m}}}$, such that the matrix*

   $$\boldsymbol{L_N} = (\boldsymbol{\eta_N}(Y=y_{N_1}) - \boldsymbol{\eta_N}(Y=y_{N_0}),\ldots,\boldsymbol{\eta_n}(Y=y_{N_{2m}}) - \boldsymbol{\eta_N}(Y=y_{N_0})) \tag{17}$$

   *of size $2m\times 2m$ is invertible.*

4. *There exist $k+1$ distinct points $\boldsymbol{y_{Z_0}},\boldsymbol{y_{Z_1}},\ldots,\boldsymbol{y_{Z_k}}$, such that the matrix*

   $$\boldsymbol{L_Z} = (\boldsymbol{\eta_Z}(Y=y_{Z_1}) - \boldsymbol{\eta_Z}(Y=y_{Z_0}),\ldots,\boldsymbol{\eta_z}(Y=y_{Z_k}) - \boldsymbol{\eta_z}(Y=y_{Z_0})) \tag{18}$$

   *of size $k\times k$ is invertible.*

5. *The function class of $W_{i,j}$ can be expressed by a Taylor series: for each $W_{i,j}$, $W(\mathbf{0})_{i,j} = 0$.*

*then the true latent causal variables $\mathbf{Z}$ are related to the estimated latent causal variables $\hat{\mathbf{Z}}$ by the following relationship: $\mathbf{Z} = \mathbf{P}\hat{\mathbf{Z}} + \mathbf{c}$, where $\mathbf{P}$ denotes the permutation matrix with scaling, $\mathbf{c}$ denotes a constant vector.*

The theorem guarantees that the causal variables can be identified up to simple linear transformations, i.e., permutation and scaling, under these assumptions. Intuitively, it shows that in the worst case, given a fixed style $s$ (fixed rotations, lighting conditions, positions, etc.), it requires $k + 1$ clean examples from distinct classes to recover the causal variables and their relations, where $k$ is the dimension of the sufficient statistics for the causal variables $\mathbf{Z}$. For the identifiability of the causal structure among latent variables $\mathbf{Z}$, the Corollary 2 of the existing paper [38] demonstrates that the causal structure can be identified up to the Markov equivalence class.

In our method, we assume the causal association among causal variables is a fully-connected directed acyclic graph, and we assume the distribution of latent noise variables is Gaussian. We can obtain $k = m + (m(m+1))/2$, where $m$ is the number of causal variables. If $n_s$ is the number of different style combinations, then $n_s(k+1)$ clean examples from distinct classes are required. This is because, for each style combination, the parameters of the generative models can generally differ, necessitating the selection of clean examples for each style combination to identify the different parameters. Given that the number of latent causal variables in our causal graph is four, $n_s \times 15$ clean examples from distinct classes are required to identify the causal model.

Note that this is the theoretical worst-case scenario under the nonlinear ICA framework. If we assume that *changes in style combinations do not affect the parameters $(\boldsymbol{f}^s, \boldsymbol{W}, \boldsymbol{\mu}_N, \boldsymbol{\sigma}_N^2)$*, then the data-generating process is invariant across different styles. Under this assumption, a best-case scenario can be derived such that we only need $k + 1$ clean examples to identify the parameters. For our causal graph, which includes four latent causal variables, only 15 clean examples from distinct classes are necessary to identify the causal model.

By demonstrating both the worst-case and best-case scenarios, and showing the assumptions required to achieve the best cases, we believe the theorem not only provides valuable insight into our methods but also enhances the understanding of previous methods for learning the data-generating process [55].

## C  More Details of the Warmup Strategy

To improve the performance of the classification network, the information of the remaining examples is exploited to train the classification network by using the semi-supervised learning method MixMatch [4]. Specifically, let the selected clean examples be the labeled examples $\mathcal{S}_X$ and the remaining examples be the unlabeled examples $\mathcal{S}_U$. The labels in the labeled examples $\mathcal{S}_X$ are refined through the output of the classification network $\boldsymbol{g}_Y$. The outputs of the classification network $\boldsymbol{g}_Y$ for unlabeled examples are used to generate guessed labels. Then, the temperature sharpening is applied to the refined labels and guessed labels on the labeled examples and unlabeled examples, respectively. After that, the labeled examples $\mathcal{S}_X$ and the unlabeled examples $\mathcal{S}_U$ are transformed into augmented labeled examples $\mathcal{S}'_X$ and augmented unlabeled examples $\mathcal{S}'_U$ by using a linear mixing. The loss function used to train the classification network is

$$\mathcal{L}_{semi} = \mathcal{L}_{\mathcal{S}_X} + \lambda_u \mathcal{L}_{\mathcal{S}_U} + \lambda_r \mathcal{L}_{\text{reg}},$$

where $\mathcal{L}_{\mathcal{S}_X}$ is the cross-entropy loss for the labeled examples; $\mathcal{L}_{\mathcal{S}_U}$ is the mean squared error the unlabeled examples; $\mathcal{L}_{\text{reg}}$ is a regularization term to prevent the model from predicting all examples to belong to a single class. These three terms are defined as follows specifically.

$$\mathcal{L}_{\mathcal{S}_X} = -\frac{1}{|\mathcal{S}'_X|} \sum_{\boldsymbol{x},\boldsymbol{p} \in \mathcal{S}'_X} \sum_i p_i \log(q_\psi(Y = i|\boldsymbol{x})),$$

$$\mathcal{L}_{\mathcal{S}_U} = \frac{1}{|\mathcal{S}'_U|} \sum_{\boldsymbol{x},\boldsymbol{p} \in \mathcal{S}'_U} \|\boldsymbol{p} - q_\psi(Y|\boldsymbol{x})\|_2^2,$$

$$\mathcal{L}_{\text{reg}} = \sum_i \frac{1}{C} \log\left(1 \Big/ \frac{C}{|\mathcal{S}'_X| + |\mathcal{S}'_U|} \sum_{\boldsymbol{x} \in \mathcal{S}_X + \mathcal{S}_U} q_\psi(Y = i|\boldsymbol{x})\right),$$

where $\boldsymbol{p}$ is the label, $q_\psi(Y|\boldsymbol{x}) := [q_\psi(Y = 1|\boldsymbol{x}), \ldots, q_\psi(Y = C|\boldsymbol{x})]^T$, and $C$ denote number of class.

# D  More Implementation Details

We use a 6-hidden-layer convolutional network as the encoder $g_{Z,S}$, and the channel sizes of corresponding feature maps are 32, 64, 128, 256, 512, and 512 for Fashion-MNIST, CIFAR-10, CIFAR-100, and CIFAR-N. We use a 5-hidden-layer convolutional network as the encoder $g_{Z,S}$, and the channel sizes of corresponding feature maps are 32, 64, 128, 256, and 512 for WebVision. A 6-hidden-layer transposed-convolutional network as the instance decoder and the channel size of corresponding feature maps are 512, 512, 256, 128, 64, and 32 for Fashion-MNIST, CIFAR-10, CIFAR-100, and CIFAR-N. A 5-hidden-layer transposed-convolutional network is used as the instance decoder, and the channel sizes of the corresponding feature maps are 512, 256, 128, 64, and 32 for WebVision. We use a three-layer MLP with the Leak ReLU activation function as the weight model to learn the weight of causal relations among the causal variables. To generate noisy labels, a three-layer MLP with the Leak ReLU activation function is used as the noisy label decoder.

The settings of hyperparameters for semi-supervised loss follow previous work [28]. The hyperparameter $\lambda_r$ is set to 1. For FashionMNIST and CIFAR-10 dataset, the hyperparameter $\lambda_u$ is set to 5, 10, 15, 20 and 25 for noise rates 10%, 20%, 30%, 40% and 50%. For CIFAR-100 dataset, the hyperparameter $\lambda_u$ is set to 10, 25, 50, 100 and 150 for noise rates 10%, 20%, 30%, 40% and 50%. For the CIFAR-N dataset, $\lambda_u$ is set to 20 for the noise type "Worst", 100 for the noise type "Noisy" and 0 for other noise types. For WebVision dataset, $\lambda_u$ is set to 0. The hyperparameters $\lambda_{ELBO}$ and $\lambda_M$ are set to 0.5 and 0.001 for synthetic datasets. For the real-world datasets, $\lambda_{ELBO}$ and $\lambda_M$ are set to 0.1 and 0.001.

---

**Algorithm 1** CSGN

---

**Input:** A noisy dataset $\tilde{\mathcal{D}}$, Warm Up epoch $T_w$, Total epoch $T_{max}$. .

1: $g_Y^1, g_Y^2 \leftarrow \text{WarmUP}(\tilde{\mathcal{D}})$;
2: **For** T $= 1, \ldots, T_w$:
3:   **For** k=1, 2:
4:     Sample $(\boldsymbol{x}, \tilde{y}) \sim \tilde{\mathcal{D}}$;
5:     $\hat{y} \leftarrow g_Y^k(\boldsymbol{x})$;
6:     Sample $(\boldsymbol{z}, \boldsymbol{s}) \sim q_\phi^k(\boldsymbol{Z}, \boldsymbol{S}|\boldsymbol{X} = \boldsymbol{x}, Y = \hat{y})$;
7:     Calculate the prior distribution $p_{\boldsymbol{W}, \boldsymbol{\mu_N}, \boldsymbol{\sigma_N^2}}^k(\boldsymbol{Z}|Y = \hat{y})$ based on $\hat{y}$;
8:     Sample $\hat{\boldsymbol{x}} \sim p_{\boldsymbol{f_X}}^k(\boldsymbol{X}|\boldsymbol{Z} = \boldsymbol{M_X} \odot \boldsymbol{z}, \boldsymbol{S} = \boldsymbol{s})$;
9:     Sample $\hat{\tilde{y}} \sim p_{\boldsymbol{f_{\tilde{Y}}}}^k(\tilde{Y}|\boldsymbol{Z} = \boldsymbol{M_{\tilde{Y}}} \odot \boldsymbol{z})$;
10:     Calculate the loss using Eq. (5) and update networks $q_\phi^k(\cdot), p_{\boldsymbol{W}, \boldsymbol{\mu_N}, \boldsymbol{\sigma_N^2}}^k(\cdot), p_{\boldsymbol{f_X}}^k(\cdot)$ and $p_{\boldsymbol{f_{\tilde{Y}}}}^k(\cdot)$;
11: **For** T $= 1, \ldots, T_{max}$:
12:   $\mathcal{S}_X, \mathcal{S}_U \leftarrow \text{Selection}(\tilde{\mathcal{D}}, g_Y^1, g_Y^2)$;
13:   $\mathcal{S}_X', \mathcal{S}_U' \leftarrow \text{MixUp}(\mathcal{S}_X, \mathcal{S}_U)$;
14:   **For** k=1, 2:
15:     Sample $(\boldsymbol{x}, \tilde{y}) \sim \tilde{\mathcal{D}}$;
16:     $\hat{y} \leftarrow g_Y^k(\boldsymbol{x})$;
17:     Sample $(\boldsymbol{z}, \boldsymbol{s}) \sim q_\phi^k(\boldsymbol{Z}, \boldsymbol{S}|\boldsymbol{X} = \boldsymbol{x}, Y = \hat{y})$;
18:     Calculate the prior distribution $p_{\boldsymbol{W}, \boldsymbol{\mu_N}, \boldsymbol{\sigma_N^2}}^k(\boldsymbol{Z}|Y = \hat{y})$ based on $\hat{y}$;
19:     Sample $\hat{\boldsymbol{x}} \sim p_{\boldsymbol{f_X}}^k(\boldsymbol{X}|\boldsymbol{Z} = \boldsymbol{M_X} \odot \boldsymbol{z}, \boldsymbol{S} = \boldsymbol{s})$;
20:     Sample $\hat{\tilde{y}} \sim p_{\boldsymbol{f_{\tilde{Y}}}}^k(\tilde{Y}|\boldsymbol{Z} = \boldsymbol{M_{\tilde{Y}}} \odot \boldsymbol{z})$;
21:     Calculate the loss using Eq. (5) and update networks $g_Y^k(\cdot), q_\phi^k(\cdot), p_{\boldsymbol{W}, \boldsymbol{\mu_N}, \boldsymbol{\sigma_N^2}}^k(\cdot), p_{\boldsymbol{f_X}}^k(\cdot)$ and
    $p_{\boldsymbol{f_{\tilde{Y}}}}^k(\cdot)$;
**Output:** The classification networks $g_Y^1, g_Y^2$.

---

# E  Pseudocode

The algorithm of the proposed method, Causal Structure for the Generation of Noisy data (CSGN), is shown in Alg. 1. To prevent the accumulation of errors from biased selection, we adopt the

approach of using two neural networks, $g_Y^1$ and $g_Y^2$, to select clean examples for each other, following the approach in previous work [13]. These neural networks model the distributions $q_{\boldsymbol{\psi}}^1(Y|\boldsymbol{X})$ and $q_{\boldsymbol{\psi}}^2(Y|\boldsymbol{X})$, respectively.

# F   Limitations

In our paper, to provide the theoretical analysis, we assume that the causal relations among the latent causal variables are linear, i.e., the causal variables are influenced by other causal variables linearly. How to identify nonlinear causal relations among the latent causal variables is still an open problem. We will continually target this problem in our future work. Moreover, to identify causal variables, similar to many existing work [13, 28, 55, 10], we also require to selected clean examples from noisy data. In future work, we will study how to reduce the required number of clean examples to achieve better accuracy with realistic assumptions. Last, our method is a generative-based method, and it requires an additional generative network, leading to more computation costs.

# G   Other Methods in Learning with Noisy Labels

Label-noise learning is a subset of learning with noisy labels. Methods in label-noise learning refer specifically to the methods that model the transition matrices. In this section, we briefly introduce other approaches in learning with noisy labels. Some algorithms [13, 28, 57, 33] select examples which likely to be correct for training. These selections are based on the memorization effect [59, 34, 3, 32, 58], which suggests deep neural networks initially memorize major patterns before progressively memorizing minor ones. In datasets containing noisy labels, correctly labeled examples often form the majority. This enables the networks to prioritize learning from these examples at the early stage of training. As a result, these examples can typically be identified by their low loss values early on. Co-Teaching [13] employs this principle to identify small-loss examples as probably clean examples. DivideMix [28] uses a Gaussian Mixture Model to separate training examples into labeled and unlabeled sets based on their training loss, with the labeled set assumed to contain correct labels and trains networks in a semi-supervised manner [4, 16]. Some methods design robust loss functions for learning with noisy labels. If a loss function $\ell$ is symmetric, i.e., $\sum_i \ell(\boldsymbol{g}_Y(\boldsymbol{X}), i) = c$, where $c$ is a constant, the loss function $\ell$ is robust to label noise [11, 12]. By combating Cross Entropy and Reverse Cross Entropy, Symmetric Cross Entropy Learning is robust to label noise [47]. Some methods correct noisy labels in datasets [46, 60]. Tanaka *et al.* proposed a method that updates network parameters and class labels alternatively. Zheng *et al.* provided theoretical guarantees for data-re-calibrating methods and proposed a label-correction algorithm with a guaranteed success rate [60]. NPC [1] is a post-processing method that models the generation process of clean labels and then uses the model to calibrate the prediction of classifiers.

# H   Sensitivity Tests for Hyper-Parameters

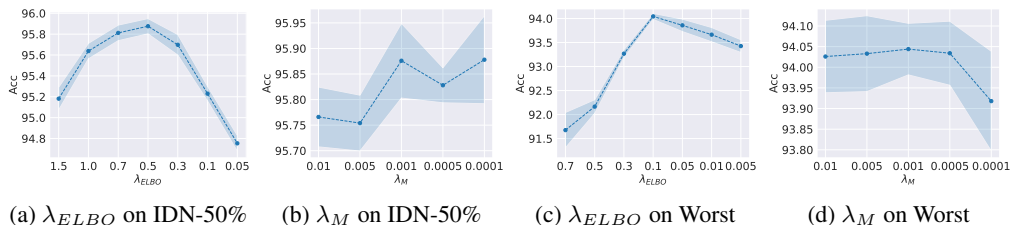

(a) $\lambda_{ELBO}$ on IDN-50%    (b) $\lambda_M$ on IDN-50%    (c) $\lambda_{ELBO}$ on Worst    (d) $\lambda_M$ on Worst

Figure 6: Illustration of the test accuracy on CIFAR-10 under instance-dependent noise with noise rate 0.5 and Worst human label set, respectively. The error bar for standard deviation has been shaded.

We conduct sensitivity tests on hyperparameters $\lambda_{ELBO}$ and $\lambda_M$ on CIFAR-10 dataset under the instance-dependent label noise with the noise rate of 0.5 and CIFAR-10N dataset under the real-world label noise, the Worst human label set. The experiment results are shown in Fig. 6. The experiment results demonstrate that the hyperparameters $\lambda_{ELBO}$ and $\lambda_M$ are not sensitive. For the datasets under

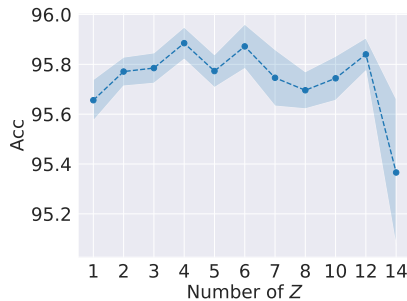

Figure 7: The test accuracy on CIFAR-10 for different numbers of causal variables under instance-dependent noise with a noise rate of 0.5. The error bar for standard deviation has been shaded.

the synthesis noise, the hyperparameters $\lambda_{ELBO}$ and $\lambda_M$ are set as 0.5 and 0.001. For the datasets under the real-world noise, the hyperparameters $\lambda_{ELBO}$ and $\lambda_M$ are set as 0.1 and 0.001.

We also conduct a sensitivity test for the number of causal variables on CIFAR-10 dataset under instance-dependent noise with a noise rate of 0.5. Experiment results are shown in Fig. 7. The results demonstrate that the proposed method is not sensitive to the number of causal variables ranging from 1 to 12. We set the number of causal variables at 4 for all experiments.

# I   Ablation Study

Table 8: Ablation study for warmup strategies. CSGN-WOSM indicates the version of CSGN without warmup using the semi-supervised learning technique. Experiments on CIFAR-10.

| | CIFAR-10 | | | | |
|---|---|---|---|---|---|
| | IDN-10% | IDN-20% | IDN-30% | IDN-40% | IDN-50% |
| CSGN-WOSM | $96.19 \pm 0.12$ | $96.05 \pm 0.07$ | $95.94 \pm 0.07$ | $95.88 \pm 0.15$ | $94.43 \pm 0.07$ |
| CSGN (ours) | $\mathbf{96.46 \pm 0.06}$ | $\mathbf{96.32 \pm 0.05}$ | $\mathbf{96.27 \pm 0.11}$ | $\mathbf{96.30 \pm 0.11}$ | $\mathbf{95.88 \pm 0.08}$ |

**Warmup Strategies**   We conduct the ablation study to assess the impact of removing the semi-supervised learning warmup phase from the CSGN method. In the study, we replaced the semi-supervised warmup with a regular early-stopping approach, where the neural networks were trained for 10 epochs on the training data. The variant of CSGN without the semi-supervised learning warmup is denoted as CSGN-WOSM. The experiment results are shown in Tab 8. The results indicate that CSGN retains its effectiveness even without the semi-supervised learning warmup.

Table 9: Ablation study for ELBO loss and $\mathcal{L}_M$. Experiments are on CIFAR-10.

| | CIFAR-10 | | | | |
|---|---|---|---|---|---|
| | IDN-10% | IDN-20% | IDN-30% | IDN-40% | IDN-50% |
| CSGN without $ELBO$ and $\mathcal{L}_M$ | $96.03 \pm 0.11$ | $95.71 \pm 0.10$ | $95.46 \pm 0.06$ | $95.44 \pm 0.12$ | $88.10 \pm 0.26$ |
| CSGN without $ELBO$ | $96.04 \pm 0.05$ | $95.75 \pm 0.08$ | $95.54 \pm 0.09$ | $95.29 \pm 0.05$ | $88.47 \pm 0.13$ |
| CSGN without $\mathcal{L}_M$ | $96.26 \pm 0.08$ | $96.17 \pm 0.05$ | $96.24 \pm 0.04$ | $96.29 \pm 0.07$ | $95.74 \pm 0.05$ |
| CSGN (ours) | $\mathbf{96.46 \pm 0.06}$ | $\mathbf{96.32 \pm 0.05}$ | $\mathbf{96.27 \pm 0.11}$ | $\mathbf{96.30 \pm 0.11}$ | $\mathbf{95.88 \pm 0.08}$ |

**Loss Functions**   To evaluate the impact of the loss functions, $ELBO$ and $\mathcal{L}_M$, we conduct an ablation study by removing these loss functions and training classification networks to observe their classification performances. The experiment results are shown in Tab. 9. The results demonstrate the effectiveness of each loss function.

**Semi-supervised Learning and Sample Selection**   We remove the semi-supervised learning technique and clean sample selection in our method. Instead, we use another simple algorithm PES [3] which trains a label-noise-robust classifier via early stopping. We leverage the prediction of this classifier for estimating the clean label. We refer to this variant of CSGN as CSGN-PES. The loss of

Table 10: CSGN works with PES. Experiments are on CIFAR-10.

| | CIFAR-10 | | | | |
|---|---|---|---|---|---|
| | IDN-10% | IDN-20% | IDN-30% | IDN-40% | IDN-50% |
| PES | $92.95 \pm 0.10$ | $92.24 \pm 0.18$ | $91.08 \pm 0.11$ | $87.82 \pm 1.11$ | $82.70 \pm 1.25$ |
| CSGN-PES | $93.05 \pm 0.06$ | $92.61 \pm 0.11$ | $91.88 \pm 0.07$ | $89.42 \pm 0.05$ | $85.45 \pm 0.05$ |
| CSGN (ours) | $\mathbf{96.46 \pm 0.06}$ | $\mathbf{96.32 \pm 0.05}$ | $\mathbf{96.27 \pm 0.11}$ | $\mathbf{96.30 \pm 0.11}$ | $\mathbf{95.88 \pm 0.08}$ |

CSGN-PES is

$$\mathcal{L}_{pes} = \mathbb{E}_{(\boldsymbol{x},\tilde{y})\sim\tilde{\mathcal{D}}}[\ell_{ce}(q_\psi(\boldsymbol{x}),\tilde{y})] - \lambda_{ELBO}ELBO + \lambda_M\mathcal{L}_M, \tag{19}$$

where $\ell_{ce}$ is the cross-entropy loss.

The experiment results are shown in Tab. 10. The experiment settings follow the settings of PES. The experiment results show that our method can still work well when without using the semi-supervised learning technique and clean sample selection.

## J    The Number and the Accuracy of the Selected Clean Samples

Table 11: The number and the accuracy of the selected clean samples on CIFAR-10.

| | CIFAR-10 | | | | |
|---|---|---|---|---|---|
| | IDN-10% | IDN-20% | IDN-30% | IDN-40% | IDN-50% |
| Number | $43707.00 \pm 3.52$ | $39911.20 \pm 14.47$ | $35040.20 \pm 5.49$ | $29913.40 \pm 4.41$ | $25016.40 \pm 5.24$ |
| Accuracy | $99.41 \pm 0.00$ | $99.05 \pm 0.03$ | $98.86 \pm 0.01$ | $98.77 \pm 0.01$ | $97.93 \pm 0.02$ |

Table 12: The number and the accuracy of the selected clean samples on CIFAR-100.

| | CIFAR-100 | | | | |
|---|---|---|---|---|---|
| | IDN-10% | IDN-20% | IDN-30% | IDN-40% | IDN-50% |
| Number | $41354.40 \pm 49.30$ | $37391.40 \pm 126.90$ | $32974.80 \pm 66.08$ | $28210.00 \pm 44.47$ | $26962.40 \pm 37.79$ |
| Accuracy | $99.74 \pm 0.01$ | $99.62 \pm 0.03$ | $99.40 \pm 0.01$ | $98.93 \pm 0.01$ | $85.58 \pm 0.12$ |

We report the number and the accuracy of the selected clean samples on CIFAR-10 and CIFAR-100 datasets. The results are shown in Tab. 11 and 12. The results indicate that our method can select a large number of clean samples while maintaining high accuracy.

## K    The Performance on the Selected Clean Samples

Table 13: Performance of the classifier trained on selected clean samples. Experiments are on CIFAR-10.

| | CIFAR-10 | | | | |
|---|---|---|---|---|---|
| | IDN-10% | IDN-20% | IDN-30% | IDN-40% | IDN-50% |
| CE-clean | $94.19 \pm 0.09$ | $93.69 \pm 0.09$ | $93.46 \pm 0.07$ | $92.16 \pm 0.08$ | $91.46 \pm 0.05$ |
| CSGN | $\mathbf{96.46 \pm 0.06}$ | $\mathbf{96.32 \pm 0.05}$ | $\mathbf{96.27 \pm 0.11}$ | $\mathbf{96.30 \pm 0.11}$ | $\mathbf{95.88 \pm 0.08}$ |

Table 14: Performance of the classifier trained on selected clean samples. Experiments are on CIFAR-100.

| | CIFAR-100 | | | | |
|---|---|---|---|---|---|
| | IDN-10% | IDN-20% | IDN-30% | IDN-40% | IDN-50% |
| CE-clean | $75.28 \pm 0.16$ | $74.66 \pm 0.15$ | $73.44 \pm 0.12$ | $71.89 \pm 0.09$ | $66.03 \pm 0.25$ |
| CSGN | $\mathbf{80.59 \pm 0.07}$ | $\mathbf{79.35 \pm 0.12}$ | $\mathbf{78.91 \pm 0.11}$ | $\mathbf{78.58 \pm 0.15}$ | $\mathbf{74.60 \pm 0.17}$ |

We also conduct experiments on the selected clean samples. Specifically, we trained a classification network using standard cross-entropy loss on these samples, employing the same architecture as our model, CSGN. The results of these experiments are presented in Tab. 13 and 14. These experiment results show that the performance of the network trained on the selected clean samples is lower than

that of our method. The primary reasons are the reduced sample size of the selected clean samples and a distribution shift from the overall dataset, i.e., the distribution of the selected clean samples is different from the whole dataset. Consequently, despite a lower noise rate in the selected clean samples, the performance of the classification network remains below that of our method.

## L  Experiments on the Synthetic Dataset

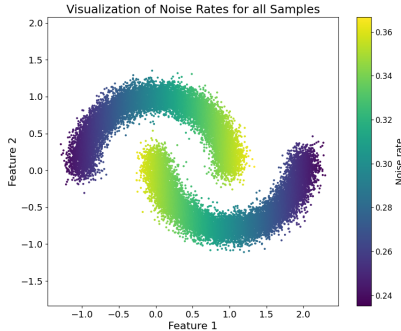

Figure 8: The visualization of the training data. The noise rate is dependent on feature 2. The average noise rate is 0.3.

To verify the proposed method can learn the causal graph and the mask, we conduct an experiment using a synthetic dataset known as the "moon dataset". The data points have two dimensions and are categorized into two distinct categories. To create noisy labels caused by a single factor, the noise rate for each data point is dependent on the value of its second dimension. The visualization of noise rates for all training samples is shown in Fig. 8. Note that the causal variables of the moon dataset are independent. We train our model on this synthetic dataset with the dimension of the latent factor $Z$ set to 2.

After training, the causal weight between two causal variables is -0.0008. The influence is small enough, which means that they are independent. The values of the mask variable $M_{\tilde{Y}}$ for noisy labels are [0.0000, 0.0232], which shows that our mask mechanism effectively identifies and selects the critical latent factor responsible for generating noisy labels.

We also compare the performance of our method with that of MEIDTM and CausalNL. MEIDTM does not model any causal mechanism. CausalNL roughly models the data-generating process, but the direct cause of noisy labels is the image, which is not aligned with the generating process of the moon dataset. Our model is the closest to the real-world data-generating process compared with these methods. Empirically, our method can achieve a test accuracy of $98.07 \pm 0.69\%$ and an estimation error for the transition matrix of $0.10 \pm 0.07$. In comparison, the test accuracy for CausalNL, which does not model the latent causal structure, is $97.88 \pm 0.75\%$ and a transition matrix estimation error of $0.12 \pm 0.06$. At the same time, the test accuracy of MEIDTM is $91.06 \pm 0.75\%$, and the estimation error of the transition matrix is $0.45 \pm 0.16$. The results show that a good causal structure can lead to good transition matrix estimation.

## M  Comparison of the Noise Transition Matrix

We use t-SNE visualization to compare the noise transition matrix learned by our method with those derived from the MEIDTM [8]. We also select 30 pairs of data points with the same predicted clean labels. The dataset is CIFAR-10 with instance-dependent label noise, and the noise rate is 50%. The experiment results are shown in Fig. 9. These data points are the same in two figures. We can see that the distance between the same pair is different in the two images. For example, the pairs with number 25 are close to each other in the first figure but are further apart in the second figure. This can verify that the similarity learned by our method is different from the instance-dependent transition matrix-based method MEIDTM.

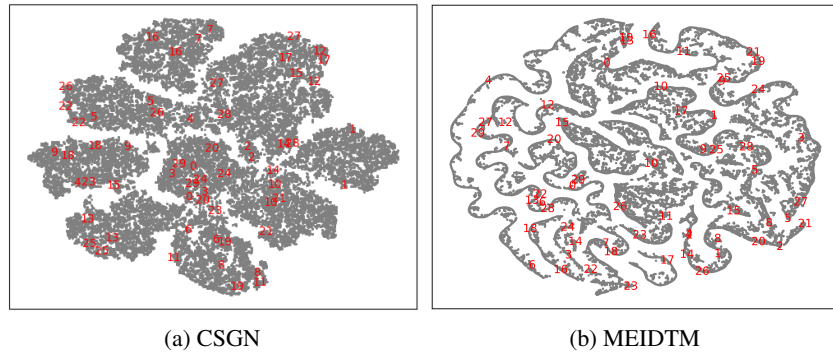

|                |                  |
|:--------------:|:----------------:|
| (a) CSGN       | (b) MEIDTM       |

Figure 9: The t-SNE visualization of the similarity of the learned noise transition. The pairs of data points with the same predicted clean label are marked with the same number. The distances between two data points represent the difference between the two noise transition matrices of these data points. The distance between the pair with the same number is different in the two images.

## N    Impact Statements

This paper presents work whose goal is to advance the field of learning with noisy labels. This work can reduce the cost (annotation costs) of training artificial intelligence models. Therefore, the threshold for using artificial intelligence has been lowered, and artificial intelligence technology is more widely used in society. There are no foreseeable negative effects.

